# Unsupervised Anomaly Detection in The Presence of Missing Values

**Feng Xiao**[1]        **Jicong Fan** [1,2*]
[1]The Chinese University of Hong Kong, Shenzhen, China
[2]Shenzhen Research Institute of Big Data, Shenzhen, China
xiaofeng.cs.ds@gmail.com    fanjicong@cuhk.edu.cn

## Abstract

Anomaly detection methods typically require fully observed data for model training and inference and cannot handle incomplete data, while the missing data problem is pervasive in science and engineering, leading to challenges in many important applications such as abnormal user detection in recommendation systems and novel or anomalous cell detection in bioinformatics, where the missing rates can be higher than 30% or even 80%. In this work, first, we construct and evaluate a straightforward strategy, "impute-then-detect", via combining state-of-the-art imputation methods with unsupervised anomaly detection methods, where the training data are composed of normal samples only. We observe that such two-stage methods frequently yield imputation bias from normal data, namely, the imputation methods are inclined to make incomplete samples "normal", where the fundamental reason is that the imputation models learned only on normal data and cannot generalize well to abnormal data in the inference stage. To address this challenge, we propose an end-to-end method that integrates data imputation with anomaly detection into a unified optimization problem. The proposed model learns to generate well-designed pseudo-abnormal samples to mitigate the imputation bias and ensure the discrimination ability of both the imputation and detection processes. Furthermore, we provide theoretical guarantees for the effectiveness of the proposed method, proving that the proposed method can correctly detect anomalies with high probability. Experimental results on datasets with manually constructed missing values and inherent missing values demonstrate that our proposed method effectively mitigates the imputation bias and surpasses the baseline methods significantly. The source code of our method is available at https://github.com/jicongfan/ImAD-Anomaly-Detection-With-Missing-Data.

## 1 Introduction

Anomaly detection (AD) [Breunig et al., 2000, Schölkopf et al., 2001, Liu et al., 2008, Pevnỳ, 2016, Zong et al., 2018, Ruff et al., 2018, Cai and Fan, 2022, Fu et al., 2024, Zhang et al., 2024, Xiao et al., 2025], aiming at identifying anomalous or novel samples in data, is a crucial machine learning problem. It finds extensive applications in many high-stakes fields such as biology, healthcare, finance, and cybersecurity. Data missing or incompleteness, a persistent and unavoidable issue in many real-world situations, often arises during the processes of data collection, transmission, and storage. Moreover, in fields like bioinformatics (e.g. single-cell RNA sequencing) [Zhang and Zhang, 2018], psychology (e.g. questionnaire data) [Schlomer et al., 2010], and recommendation systems (e.g. user-item interaction data) [Shani and Gunawardana, 2011, Fan et al., 2024], the data missing rates are often higher than $30\%$ or even $80\%$. Indeed, the missing data problems lead to

---

many challenges for anomaly detection, such as detecting anomalous cells or rare cell types based on incomplete single-cell RNA sequencing data [Fa et al., 2021] and identifying abnormal users in recommendation systems [Yang and Cai, 2017]. Regrettably, most existing AD methods necessitate complete data in both the training and test sets, rendering them ill-equipped to handle datasets with missing values. Consequently, addressing the AD challenge in the context of incomplete data becomes both necessary and inevitable.

A naive strategy is filling the missing values by statistical characteristics such as mean or median and then performing anomaly detection. Taking two real-world datasets "Adult" and "KDD" as examples, we consider the mechanism missing completely at random and fill the missing entries with the variable means and then perform a classical AD methods Isolation Forest [Liu et al., 2008]) and two deep learning based AD methods (Deep SVDD [Ruff et al., 2018] and NeutraL AD [Qiu et al., 2021]). The results are shown in Figure 1. The detection accuracies of the four methods degrade signifi-

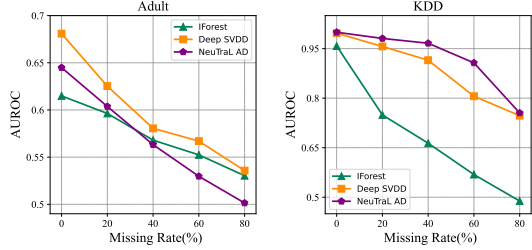

Figure 1: Performance (AUROC) degradation of anomaly detection methods with increasing missing rate on Adult and KDD datasets.

cantly with the missing rate increases. This verified the failure of the naive strategy and the difficulty of unsupervised anomaly detection with missing values. Besides the naive imputation, one may consider using more powerful imputation algorithms [Dempster et al., 1977, Pigott, 2001, Candes and Recht, 2012, Stekhoven and Bühlmann, 2012, Gondara and Wang, 2018, Yoon et al., 2018, Fan et al., 2020, Muzellec et al., 2020] to recover the missing values and subsequently implementing AD algorithms on imputed data. We refer to this strategy as "impute-then-detect".

It is worth noting that, for unsupervised anomaly detection, where the training set is composed of only normal samples, such "impute-then-detect" methods would yield imputation bias for normal data, i.e., the imputation methods are inclined to recover an abnormal sample with missing values as "normal" as possible during the inference, which leads to lower recall or higher false negative rate. Figure 2 clearly shows the negative impacts of the imputation bias, which is worth studying and addressing. The main challenge is that the training set and test set do not satisfy the condition of identical distribution and the imputation model trained only on incomplete normal data does not generalize well to incomplete ab-

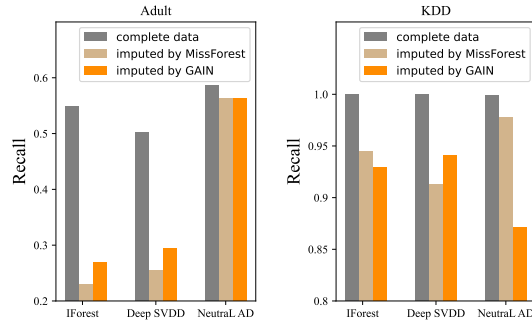

Figure 2: The degradation of recall rate of abnormal data on "impute-then-detect" methods.

normal data. In Section 4.2, we quantitatively and comprehensively evaluate the "impute-then-detect" methods using state-of-the-art imputation algorithms and AD algorithms.

To tackle the aforementioned problem, in this paper, we propose an end-to-end method, called ImAD, for unsupervised anomaly detection on incomplete data. The main idea of ImAD is to integrate data **im**putation and **a**nomaly **d**etection into a unified optimization objective and alleviate imputation bias by automatically learning to generate pseudo-abnormal samples. Note that the pseudo-abnormal samples are by-products of the training process and we do not use any extra data in all experiments. Our contributions are summarized as follows.

- We study the imputation bias problem of the "impute-then-detect" pipeline and quantitatively evaluate their detection performance.
- We propose a novel method ImAD for AD on incomplete data. To the best of our knowledge, it is the first end-to-end unsupervised AD method in the presence of missing value.
- We provide theoretical guarantees for ImAD, proving that it can correctly detect anomalies with high probability.
- We compare ImAD with more than 9 baselines on 11 real datasets of various domains, covering datasets with manually constructed missing values and datasets with inherent missing values.

## 2 Related Work

**Missing Data Imputation**    Data imputation fills missing data with plausible values and provides imputed data for downstream tasks such as classification, clustering, and visualization. As the missing data problem is prevalent in many fields, the study on missing data imputation is extensive, and many algorithms have been proposed in the past decades. Mayer et al. [2019] pointed out that there are approximately 150 implementations available to handle missing data. These methods can be roughly organized into three categories. The first category is based on the iterative regression model, such as well-known methods Multiple Imputation by Chained Equations (MICE) [Royston and White, 2011] and MissForest [Stekhoven and Bühlmann, 2012] that trains random forests on observed data through an iterative imputation scheme. The second category is the matrix completion methods [Candes and Recht, 2012, Mazumder et al., 2010, Fan and Chow, 2018, Fan et al., 2019, 2020]. The third category is based on deep learning especially deep generative models [Fan and Chow, 2017, Yoon et al., 2018, Li et al., 2019, Muzellec et al., 2020]. For instance, Yoon et al. [2018] proposed generative adversarial imputation network (GAIN) based on generative adversarial network (GAN) [Goodfellow et al., 2014] and [Tashiro et al., 2021] proposed conditional score-based diffusion models for probabilistic time-series imputation (CSDI) based diffusion model [Sohl-Dickstein et al., 2015]. Indeed, these deep imputation methods often achieve state-of-the-art performance in the tasks of missing data imputation, when the distributions of the training data and testing data are identical. However, their performance in recovering the missing values for unsupervised AD is rarely studied.

**Anomaly detection on incomplete data**    The research on anomaly detection in the presence of missing values is very limited. To the best of the authors' knowledge, [Zemicheal and Dietterich, 2019] is the first work evaluating the detection performance of anomaly detection methods combined with different data imputation techniques. Their experiments of anomaly detection on a few UCI datasets with missing values showed that implementations of unsupervised anomaly detection methods such as Isolation Forest [Liu et al., 2008] on incomplete data should always include algorithms for handling missing values and the imputation contributes to improving the detection performance of anomaly methods. Fan et al. [2022] studied the problem of statistical process monitoring with missing values and proposed a fast incremental nonlinear matrix completion method for online and sequential imputation. Sarda et al. [2023] provided a study of existing unsupervised anomaly detection methods on GAN-imputed data.

It's worth noting that the strategies used in [Zemicheal and Dietterich, 2019, Fan et al., 2022, Sarda et al., 2023] are two-stage methods, where the imputation models are trained on the training dataset that does not contain any abnormal data or only contains very few unlabeled outliers. As a result, the imputation model will not generalize well on abnormal data during the inference and will use the learned pattern of normal data to fill the missing values of abnormal data, which makes the abnormal data similar to normal data and hence lowers detection accuracy. In contrast, our method integrates data imputation and anomaly detection into a unified process, and alleviates the imputation bias via introducing pseudo-abnormal samples, and hence achieves superior detection accuracy.

## 3 Proposed Method

### 3.1 Problem Formulation and Our Motivation

Given $n$ samples $\mathbf{x}_1, \mathbf{x}_2, \cdots, \mathbf{x}_n$ drawn from an unknown distribution $\mathcal{D}_{\mathbf{x}} \subseteq \mathbb{R}^m$, the goal of unsupervised AD is to learn a decision function $f : \mathbb{R}^m \rightarrow \{0, 1\}$ by utilizing only these $n$ samples, such that $f(\mathbf{x}) = 0$ if $\mathbf{x} \in \mathcal{D}_{\mathbf{x}}$ and $f(\mathbf{x}) = 1$ if $\mathbf{x} \notin \mathcal{D}_{\mathbf{x}}$. We consider the scenario that $\mathbf{X} := [\mathbf{x}_1^\top, \mathbf{x}_2^\top, \cdots, \mathbf{x}_n^\top]^\top \in \mathbb{R}^{n \times m}$ contains missing values. Let $\mathbf{M} \in \{0, 1\}^{n \times m}$ be a mask matrix determined by some missing mechanism $\mathcal{M}$ such as MCAR, MAR, or MNAR, where $M_{i,j} = 1$ means $X_{i,j}$ is observed and $M_{i,j} = 0$ means $X_{i,j}$ is missing. Then the observed incomplete matrix is

$$\check{\mathbf{X}} = [\check{\mathbf{x}}_1^\top, \check{\mathbf{x}}_2^\top, \cdots, \check{\mathbf{x}}_n^\top]^\top = \mathcal{M}(\mathbf{X}) = \mathbf{X} \odot \mathbf{M} \tag{1}$$

where $\odot$ is the Hadamard product. Equation (1) implies that the missing values of $\mathbf{X}$ are temporarily filled with zeros. In many scenarios such as gene expression data analysis, recommendation systems, and questionnaire surveys, the data missing rate in $\check{\mathbf{X}}$ is often high. Training an anomaly detection model $f$ on $\check{\mathbf{X}}$ and using it to detect anomalies in new incomplete data has practical significance

such as detecting anomalous cells or rare cell types in bioinformatics, identifying abnormal users in recommendation systems, and recognizing unusual subjects using questionnaires of psychology.

As mentioned before, conventional AD methods are vulnerable to missing values and a good imputation algorithm can raise the detection accuracy of an anomaly detection method to some extent. However, the strategy "impute-then-detect" is inclined to make incomplete abnormal samples normal and hence cannot provide satisfactory detection performance. Therefore, in this work, we aim to provide an end-to-end unsupervised anomaly detection method in the presence of missing values to mitigate the imputation bias and improve the detection accuracy. The most challenging problem is that the imputation model (denoted as $\mathcal{I}$) trained only on incomplete normal data cannot generalize well to incomplete abnormal data. To solve the challenge, we take the following strategy and consideration.

We propose to learn a model that can generate some pseudo-abnormal samples, and then learn an imputation model from both the original normal data and the generated pseudo-abnormal samples. Thus, the learned imputation model can generalize well to incomplete abnormal data during inference and recover the missing values with high accuracy, which further improves the accuracy of anomaly detection. However, we encounter the following issues.

- It is non-trivial to generate meaningful pseudo-abnormal samples that are similar enough to real ones. The reason is that the distribution (i.e., $\mathcal{D}_{\mathbf{x}}$) of training data is unknown and the data dimension $m$ is often high.
- The incompleteness of $\mathbf{X}$ further increases the difficulty of generating pseudo-abnormal samples.
- On the other hand, the generated pseudo-abnormal samples should not be too far from the normal data, where a large gap will make the learned imputation model fail to impute the abnormal samples close to normal data and cause the abnormal samples to be hard to detect.
- The generating model, imputation model, and detection model should be coordinated with each other and as a whole to ensure the reliability of the inference.

## 3.2 Learning Framework of ImAD

To address the aforementioned challenges, we propose to find a $d$-dimensional latent space $\mathcal{Z}$ where the normal data are lying and then generate pseudo-abnormal samples around the normal samples in $\mathcal{Z}$. The samples in $\mathcal{Z}$ will be mapped back by a neural network to the original data space, yielding reliable pseudo-abnormal data.

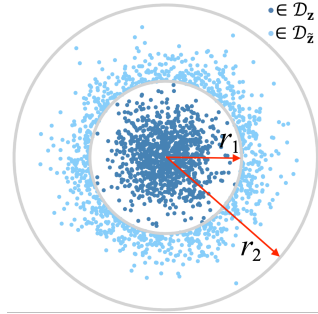

We define $\mathcal{D}_{\mathbf{z}}$ as the latent distribution of the normal data in $\mathcal{Z}$ and define $\mathcal{D}_{\tilde{\mathbf{z}}}$ as the latent distribution of pseudo-abnormal data in $\mathcal{Z}$. Since the patterns of normality are limited and the patterns of abnormality are unlimited, we let $\mathcal{D}_{\mathbf{z}}$ be a truncated Gaussian distribution (a hyperball denoted by $\mathcal{B}$, with radius $r_1$) in $\mathcal{Z}$ and assume that the remaining region of $\mathcal{Z}$ is the abnormal region, denoted as $\mathcal{Z} \setminus \mathcal{B}$. It should be pointed out that there is no need to define $\mathcal{D}_{\tilde{\mathbf{z}}}$ in the entire space $\mathcal{Z} \setminus \mathcal{B}$, which will be explained in the discussion for Theorem 3.2(b) and further supported by Theorem 3.4 in Section

Figure 3: Visualization of $\mathcal{D}_{\mathbf{z}}$ and $\mathcal{D}_{\tilde{\mathbf{z}}}$ in 2-D latent space $\mathcal{Z}$.

3.4. Instead, we only need to define $\mathcal{D}_{\tilde{\mathbf{z}}}$ in a small region of $\mathcal{Z} \setminus \mathcal{B}$ that encloses $\mathcal{B}$, which will reduce the uncertainty of random sampling (or samples size equivalently) and make it easier for mapping the samples back to the original data space. Thus, we define $\mathcal{D}_{\tilde{\mathbf{z}}}$ as a hypershell surrounding $\mathcal{B}$ and let $\mathcal{D}_{\tilde{\mathbf{z}}}$ be a truncated Gaussian. The radii of the two hyperspheres forming the hypershell are $r_1$ and $r_2$ respectively, where $r_2 > r_1$. An illustration of $\mathcal{D}_{\mathbf{z}}$ and $\mathcal{D}_{\tilde{\mathbf{z}}}$ in 2-D space is shown in Figure 3, where $\mathcal{D}_{\mathbf{z}}$ and $\mathcal{D}_{\tilde{\mathbf{z}}}$ are truncated Gaussian from $\mathcal{N}(\mathbf{0}, 0.5^2 \cdot \mathbf{I}_2)$ and $\mathcal{N}(\mathbf{0}, \mathbf{I}_2)$ respectively. The theoretical analysis for sampling from $\mathcal{D}_{\mathbf{z}}$ and $\mathcal{D}_{\tilde{\mathbf{z}}}$ is in Appendix A. We learn a reconstructor $\mathcal{R} : \mathbb{R}^d \rightarrow \mathbb{R}^m$ to transform the samples drawn from $\mathcal{D}_{\mathbf{z}}$ to the original data distribution $\mathcal{D}_{\mathbf{x}}$, i.e.,

$$\mathcal{D}_{\mathbf{x}} \approx \mathcal{R}(\mathcal{D}_{\mathbf{z}}). \tag{2}$$

$\mathcal{R}$ is actually a reconstruction model that recovers the original data from the latent space $\mathcal{Z}$. With $\mathcal{D}_{\tilde{\mathbf{z}}}$ and $\mathcal{R}$, we can obtain a distribution $\mathcal{D}_{\tilde{\mathbf{x}}}$ of pseudo-abnormal data in the original data space as

$$\mathcal{D}_{\tilde{\mathbf{x}}} := \mathcal{R}(\mathcal{D}_{\tilde{\mathbf{z}}}). \tag{3}$$

The samples (denoted by $\tilde{\mathbf{x}}$) drawn from $\mathcal{D}_{\tilde{\mathbf{x}}}$ are reasonable pseudo-abnormal samples, which will be explained by the discussion for Theorem 3.2(a) in Section 3.4.

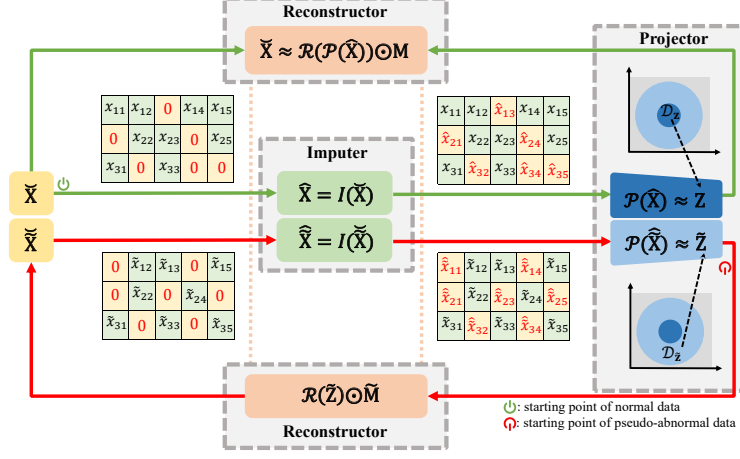

**Figure 4:** ImAD framework. $\breve{\mathbf{X}}$ and $\breve{\tilde{\mathbf{X}}}$ denote the normal and pseudo-abnormal data with missing values, respectively, while $\hat{\mathbf{X}}$ and $\hat{\tilde{\mathbf{X}}}$ are the corresponding imputed data.

Now we use a projector $\mathcal{P} : \mathbb{R}^m \to \mathbb{R}^d$ to transform $\mathcal{D}_{\mathbf{x}}$ and $\mathcal{D}_{\tilde{\mathbf{x}}}$ into $\mathcal{D}_{\mathbf{z}}$ and $\mathcal{D}_{\tilde{\mathbf{z}}}$ respectively, i.e.,

$$\mathcal{D}_{\mathbf{z}} \approx \mathcal{P}(\mathcal{D}_{\mathbf{x}}), \quad \mathcal{D}_{\tilde{\mathbf{z}}} \approx \mathcal{P}(\mathcal{D}_{\tilde{\mathbf{x}}}). \tag{4}$$

However, the training set $\breve{\mathbf{X}} = \mathcal{M}(\mathbf{X})$ is incomplete, and we need to learn an imputation model $\mathcal{I}$ to recover the missing values, i.e., $\hat{\mathbf{X}} = \mathcal{I}(\breve{\mathbf{X}})$. More generally, we denote

$$\mathcal{D}_{\hat{\mathbf{x}}} = \mathcal{I}(\mathcal{D}_{\breve{\mathbf{x}}}). \tag{5}$$

We hope that the imputation model is also able to recover the missing values of the generated pseudo-abnormal samples if they have, though they are complete. We thus remove some values of the generated pseudo-abnormal samples $\tilde{\mathbf{x}} \sim \mathcal{D}_{\tilde{\mathbf{x}}}$ using missing mechanism $\tilde{\mathcal{M}}$ and let $\mathcal{D}_{\breve{\tilde{\mathbf{x}}}} = \tilde{\mathcal{M}}(\mathcal{D}_{\tilde{\mathbf{x}}})$.

The missing values are then recovered by

$$\mathcal{D}_{\hat{\tilde{\mathbf{x}}}} = \mathcal{I}(\mathcal{D}_{\breve{\tilde{\mathbf{x}}}}). \tag{6}$$

This step mitigates the problem of imputation bias encountered by the "impute-then-detect" methods.

Let $\mathcal{E}_I$, $\mathcal{E}_P$, and $\mathcal{E}_R$ denote some distance or discrepancy measure between distributions. We here show how to achieve the goals of (2), (3), (4), (5), and (6) in a unified optimization problem. First, for normal data, we solve

$$\underset{\mathcal{I},\mathcal{P},\mathcal{R}}{\text{minimize}} \; \mathcal{E}_I(\mathcal{I}(\mathcal{D}_{\breve{\mathbf{x}}}), \mathcal{D}_{\breve{\mathbf{x}}} \mid \mathcal{M}) + \mathcal{E}_P(\mathcal{P}(\mathcal{D}_{\hat{\mathbf{x}}}), \mathcal{D}_{\mathbf{z}}) + \mathcal{E}_R(\mathcal{R}(\mathcal{P}(\mathcal{D}_{\hat{\mathbf{x}}})), \mathcal{D}_{\breve{\mathbf{x}}} \mid \mathcal{M}) \tag{7}$$

For the generated pseudo-abnormal data, we solve

$$\underset{\mathcal{I},\mathcal{P},\mathcal{R}}{\text{minimize}} \; \mathcal{E}_I\big(\mathcal{I}(\tilde{\mathcal{M}}(\mathcal{R}(\mathcal{D}_{\tilde{\mathbf{z}}}))), \tilde{\mathcal{M}}(\mathcal{R}(\mathcal{D}_{\tilde{\mathbf{z}}})) \mid \tilde{\mathcal{M}}\big) + \mathcal{E}_P\big(\mathcal{P}(\mathcal{I}(\tilde{\mathcal{M}}(\mathcal{R}(\mathcal{D}_{\tilde{\mathbf{z}}})))), \mathcal{D}_{\tilde{\mathbf{z}}}\big) \tag{8}$$

Let $\hat{\mathcal{E}}.$ be a finite-sample estimation of $\mathcal{E}.$. Combining (7) and (8), we obtain the objective of ImAD:

$$\underset{\mathcal{I},\mathcal{P},\mathcal{R}}{\text{minimize}} \; \underbrace{\hat{\mathcal{E}}_I(\mathcal{I}([\breve{\mathbf{X}}; \breve{\tilde{\mathbf{X}}}]), [\breve{\mathbf{X}}; \breve{\tilde{\mathbf{X}}}] \mid [\mathbf{M}, \tilde{\mathbf{M}}])}_{\mathcal{L}^{(\text{DI})}} + \underbrace{\hat{\mathcal{E}}_P(\mathcal{P}([\hat{\mathbf{X}}; \hat{\tilde{\mathbf{X}}}]), [\mathbf{Z}; \tilde{\mathbf{Z}}])}_{\mathcal{L}^{(\text{AD})}} + \underbrace{\hat{\mathcal{E}}_R(\mathcal{R}(\mathcal{P}(\hat{\mathbf{X}})), \breve{\mathbf{X}} \mid \mathbf{M})}_{\mathcal{L}^{(\text{RE})}} \tag{9}$$

where $\breve{\tilde{\mathbf{X}}} = \mathcal{R}(\tilde{\mathbf{Z}}) \odot \tilde{\mathbf{M}}$, $\hat{\tilde{\mathbf{X}}} = \mathcal{I}(\breve{\tilde{\mathbf{X}}})$, and $[\cdot; \cdot]$ denotes the row-wise concatenation of two matrices. In (9), the samples in $\mathbf{Z}$ are drawn from $\mathcal{D}_{\mathbf{z}}$ and the samples in $\tilde{\mathbf{Z}}$ are drawn from $\mathcal{D}_{\tilde{\mathbf{z}}}$. The roles of the three parts of the objective function in (9) are analyzed as follows.

- $\mathcal{L}^{(\text{DI})}$ denotes the data imputation loss. With this loss, the imputation model will be able to recover the missing values of normal data and abnormal data.

- $\mathcal{L}^{(\text{AD})}$ denotes the anomaly detection loss. With this loss, the anomaly detection model will be discriminative and be able to project normal data and abnormal data into different regions in $\mathcal{Z}$.
- $\mathcal{L}^{(\text{RE})}$ denotes the reconstruction loss. This loss is to ensure that $\mathcal{D}_{\mathbf{z}}$ and $\mathcal{D}_{\tilde{\mathbf{z}}}$ are meaningful.

We see that our method ImAD couples data imputation with anomaly detection to a unified optimization objective. Figure 4 depicts the overall framework of ImAD, where the green and red arrows show the flow paths of normal data (starting from $\breve{\mathbf{X}}$) and pseudo-abnormal data (starting from $\tilde{\mathbf{Z}}$) respectively. The reconstructors in Figure 4 share parameters.

### 3.3 Specific Implementation of ImAD

We use three neural networks $h_\psi$, $f_\theta$ and $g_\phi$ with parameters $\psi, \theta, \phi$ to model $\mathcal{I}, \mathcal{P}$ and $\mathcal{R}$ respectively. For $\mathcal{E}$, we consider two different cases. If the samples are pair-wise, we directly use the square loss, which is simple and efficient. Thus, in $\mathcal{L}^{\text{DI}}$ and $\mathcal{L}^{\text{RE}}$, we use the square loss, and the square loss for $\mathcal{L}^{\text{RE}}$ is masked by $\mathbf{M}$. When the samples are not pair-wise, we take advantage of the Sinkhorn distance [Cuturi, 2013] derived from the optimal transport theory. The Sinkhorn distance between two distributions $\mathcal{D}_{\mathcal{U}}$ and $\mathcal{D}_{\mathcal{V}}$ supported by their finite samples $\mathcal{U} = \{\mathbf{u}_1, \mathbf{u}_2, \cdots, \mathbf{u}_{n_u}\} \sim \mathcal{D}_u$ and $\mathcal{V} = \{\mathbf{v}_1, \mathbf{v}_2, \cdots, \mathbf{v}_{n_v}\} \sim \mathcal{D}_v$ is defined as

$$\text{Sinkhorn}(\mathcal{U}, \mathcal{V}) := \min_{\mathbf{P}} \ \langle \mathbf{P}, \mathbf{C} \rangle_F + \eta \sum_{i,j} \mathbf{P}_{ij} \log(\mathbf{P}_{ij}), \qquad \text{s.t. } \mathbf{P1} = \mathbf{a}, \mathbf{P}^T \mathbf{1} = \mathbf{b}, \mathbf{P} \geq 0$$

(10)

where $\mathbf{P} \in \mathbb{R}^{n_u \times n_v}$ is the transport plan and $\mathbf{C} \in \mathbb{R}^{n_u \times n_v}$ is the metric cost matrix. The two probability vectors $\mathbf{a}$ and $\mathbf{b}$ satisfy $\mathbf{a}^T \mathbf{1} = 1, \mathbf{b}^T \mathbf{1} = 1$, and $\eta \geq 0$ is a trade-off between the Wasserstein distance and entropy regularization.

By applying $h_\psi, f_\theta, g_\phi$, square loss, and Sinkhorn distance to (9), we obtain the following problem:

$$\underset{\psi,\theta,\phi}{\text{minimize}} \ \underbrace{\text{Sinkhorn}(f_\theta(h_\psi(\breve{\mathbf{X}})), \mathbf{Z}) + \alpha \|\tilde{\mathbf{Z}} - f_\theta(h_\psi(g_\phi(\tilde{\mathbf{Z}}) \odot \tilde{\mathbf{M}}))\|_F^2}_{\mathcal{L}^{(\text{AD})}}$$
$$+ \underbrace{\beta \|([\breve{\mathbf{X}}; \breve{\tilde{\mathbf{X}}}] - h_\psi([\breve{\mathbf{X}}; \breve{\tilde{\mathbf{X}}}])) \odot [\mathbf{M}; \tilde{\mathbf{M}}]\|_F^2}_{\mathcal{L}^{(\text{DI})}} + \underbrace{\lambda \|(\breve{\mathbf{X}} - g_\phi(f_\theta(h_\psi(\breve{\mathbf{X}})))) \odot \mathbf{M}\|_F^2}_{\mathcal{L}^{(\text{RE})}}$$

(11)

Solving the problem (11), we get well trained imputer $h_{\psi^*}$ and projector $f_{\theta^*}$. For a new sample $\breve{\mathbf{x}}_{\text{new}}$ containing missing values, we define an anomaly score $s(\cdot)$ by

$$s(\breve{\mathbf{x}}_{\text{new}}) = \|f_{\theta^*}(h_{\psi^*}(\breve{\mathbf{x}}_{\text{new}}))\|,$$

(12)

which is the distance to the origin in the latent space. If $s(\breve{\mathbf{x}}_{\text{new}}) > r_1$, $\breve{\mathbf{x}}_{\text{new}}$ is detected as abnormal. Otherwise, $\breve{\mathbf{x}}_{\text{new}}$ is treated as a normal sample.

### 3.4 Theoretical Guarantees for ImAD

WLOG, we assume $f_\theta$, $g_\phi$, and $h_\psi$ all have $L$ layers, where $\theta = \{\mathbf{W}_1^f, \mathbf{W}_2^f, \ldots, \mathbf{W}_L^f\}$, $\phi = \{\mathbf{W}_1^g, \mathbf{W}_2^g, \ldots, \mathbf{W}_L^g\}$, and $\psi = \{\mathbf{W}_1^h, \mathbf{W}_2^h, \ldots, \mathbf{W}_L^h\}$. Denote the spectral norm and $\ell_{2,1}$-norm of a matrix as $\|\cdot\|_\sigma$ and $\|\cdot\|_{2,1}$ respectively. We also make the following assumptions.

**Assumption 3.1.** For $f_\theta$, $g_\phi$, and $h_\psi$, the following conditions hold: 1) $\|\mathbf{W}_l^f\|_\sigma \leq \alpha_f$, $\|\mathbf{W}_l^g\|_\sigma \leq \alpha_g$, $\|\mathbf{W}_l^h\|_\sigma \leq \alpha_h$, $\forall l \in [L]$; 2) $\|\mathbf{W}_l^f\|_{2,1} \leq b_f$, $\|\mathbf{W}_l^g\|_{2,1} \leq b_g$, $\|\mathbf{W}_l^h\|_{2,1} \leq b_h$, $\forall l \in [L]$; 3) all activation functions in $f_\theta$, $g_\phi$, and $h_\psi$ are $\rho$-Lipschitz continuous; 4) the maximum width of the layers in $f_\theta$, $g_\phi$, and $h_\psi$ is $\bar{d}$.

The following theorem can be used to obtain some deterministic guarantee for ImAD.

**Theorem 3.2.** *Under Assumption 3.1, we have:*
*(a)* $\|g_\phi(\mathbf{z}) - g_\phi(\tilde{\mathbf{z}})\| \leq \rho^L \alpha_g^L \|\mathbf{z} - \tilde{\mathbf{z}}\|$ *holds for any* $\mathbf{z}, \tilde{\mathbf{z}}$;
*(b)* $\|f_\theta(h_\psi(\breve{\mathbf{x}})) - f_\theta(h_\psi(\breve{\tilde{\mathbf{x}}}))\| \leq \rho^{2L} \alpha_f^L \alpha_h^L \|\breve{\mathbf{x}} - \breve{\tilde{\mathbf{x}}}\|$ *holds for any* $\breve{\mathbf{x}}$ *and* $\breve{\tilde{\mathbf{x}}}$.

Theorem 3.2(a) indicates that in the latent space $\mathcal{Z}$, if an abnormal sample $\tilde{\mathbf{z}} \sim \mathcal{D}_{\tilde{\mathbf{z}}}$ is close to a normal sample $\mathbf{z} \sim \mathcal{D}_{\mathbf{z}}$, in the original data space, the corresponding abnormal sample $\tilde{\mathbf{x}}$ is still close to the normal sample $\mathbf{x}$ provided that $\alpha_g$ is not too large. This means the generated pseudo-abnormal

samples are practical and useful. For Theorem 3.2(b), let's consider an incomplete abnormal sample $\breve{\tilde{\mathbf{x}}}$ and assume that its closest incomplete pseudo-abnormal sample generated by the $\tilde{\mathbf{z}}$ on the outer hypersphere (shown in Figure 3) is $\breve{\tilde{\mathbf{x}}}^*$, where $\|\breve{\tilde{\mathbf{x}}} - \breve{\tilde{\mathbf{x}}}^*\| = \beta$. Then in the latent space, we have $\|\tilde{\mathbf{z}} - \tilde{\mathbf{z}}^*\| \leq \rho^{2L}\alpha_f^L\alpha_h^L\beta$. Let the radii of the inner and outer hyperspheres be $r_1$ and $r_2$ respectively. Now we can conclude that if $r_2 - r_1 > \rho^{2L}\alpha_f^L\alpha_h^L\beta$, $\tilde{\mathbf{z}}$ is outside the decision region given by the inner hypersphere and hence $\breve{\tilde{\mathbf{x}}}$ is successfully detected as an abnormal sample.

Now we study the theoretical guarantees for our ImAD in the sense of expectation. Let $r_1$ be the thresholds for the anomaly score defined by (12) to determine whether a sample is normal or not. Let $r_2$ be the radius of the outer hypersphere enclosing $\mathcal{D}_{\tilde{\mathbf{z}}}$. Let $\bar{s}_{\breve{\mathbf{x}}}$ be the average anomaly score of the (incomplete) normal training data, i.e., $\bar{s}_{\breve{\mathbf{x}}}^2 = \frac{1}{n}\sum_{i=1}^n s(\breve{\mathbf{x}}_i)^2$. Let $\bar{\varepsilon}_{\breve{\tilde{\mathbf{x}}}}^2 := \frac{1}{n}\sum_{i=1}^n |r_2^2 - s(\breve{\tilde{\mathbf{x}}}_i)^2|$, where $\breve{\tilde{\mathbf{x}}}_i$ are the (incomplete) pseudo-abnormal samples generated during the training stage. With these definitions and Assumption 3.1, the following (proved in Appendix C) presents the theoretical generalization ability of our ImAD.

**Theorem 3.3.** *Suppose the squared anomaly score $s(\breve{\mathbf{x}})^2$ of normal data is always upper-bounded by $\gamma$, $|r_2^2 - s(\breve{\tilde{\mathbf{x}}})^2|$ of the pseudo-abnormal data is always upper-bounded by $\tilde{\gamma}$, and the absolute output of $f_\theta$ is always upper-bounded by $\vartheta$. Suppose the samples in $\breve{\mathbf{X}}$ and $\breve{\tilde{\mathbf{X}}}$ are independently drawn $\mathcal{D}_{\breve{\mathbf{x}}}$ and $\mathcal{D}_{\breve{\tilde{\mathbf{x}}}}$ respectively. Define $\kappa = \alpha_f^L\alpha_h^L$, $\zeta = \left(1 + L\left(\frac{b_f}{\alpha_f}\right)^{2/3} + L\left(\frac{b_h}{\alpha_h}\right)^{2/3}\right)^{3/2}$, $\Delta = r_1^2 - \bar{s}_{\breve{\mathbf{x}}}^2$, and $\tilde{\Delta} = r_2^2 - r_1^2 - \bar{\varepsilon}_{\breve{\tilde{\mathbf{x}}}}^2$.*
*(a) For normal data from $\mathcal{D}_{\breve{\mathbf{x}}}$, over the randomness of $\breve{\mathbf{X}}$,*

$$\mathbb{P}\left[\mathbb{E}_{\mathcal{D}_{\breve{\mathbf{x}}}}[s(\breve{\mathbf{x}})] > r_1\right] \leq \delta, \tag{13}$$

*where $\delta = 2\exp\left(-2n\left(\Delta - \frac{8\gamma + 48R\ln n}{n}\right)^2/(9\gamma^2)\right)$ and $R = \rho^{2L-1}\vartheta\kappa\zeta\|\breve{\mathbf{X}}\|_F\sqrt{d\ln(2\bar{d}^2)}$.*
*(b) For abnormal data from $\mathcal{D}_{\breve{\tilde{\mathbf{x}}}}$, over the randomness of $\breve{\tilde{\mathbf{X}}}$,*

$$\mathbb{P}\left[\mathbb{E}_{\mathcal{D}_{\breve{\tilde{\mathbf{x}}}}}[s(\breve{\tilde{\mathbf{x}}})] \geq r_1\right] \geq 1 - \tilde{\delta}, \tag{14}$$

*where $\tilde{\delta} = 2\exp\left(-2n\left(\tilde{\Delta} - \frac{8\tilde{\gamma} + 48\tilde{R}\ln n}{n}\right)^2/(9\tilde{\gamma}^2)\right)$ and $\tilde{R} = \rho^{2L-1}\vartheta\kappa\zeta\|\breve{\tilde{\mathbf{X}}}\|_F\sqrt{d\ln(2\bar{d}^2)}$.*

Theorem 3.3(a) means that a normal sample, in expectation, is detected as anomalous with probability at almost $\delta$, where $\delta$ is close to zero under some mild conditions such as $L$ is not too large and $n$ is not too small. In other words, a false alarm happens with low probability. Theorem 3.3 (b) means that an abnormal sample drawn from $\mathcal{D}_{\breve{\tilde{\mathbf{x}}}}$, in expectation, can be successfully detected with probability at least $1 - \delta$, where $\delta$ is close to zero under some mild conditions. Theorem 3.3(b) also indicates that a larger $r_2$ is better. It is worth noting that here we only focus on $\mathcal{D}_{\breve{\tilde{\mathbf{x}}}}$, which is defined by $\mathcal{D}_{\tilde{\mathbf{z}}}$, $\tilde{\mathbf{M}}$, and $g_\phi$. $\mathcal{D}_{\breve{\tilde{\mathbf{x}}}}$ can be regarded as a distribution of difficult anomalous data that are close to normal data. The anomalous samples drawn from space out of $\mathcal{D}_{\breve{\tilde{\mathbf{x}}}}$ are much easier to detect, which is further supported by the following theorem (proved in Appendix D).

**Theorem 3.4.** *Let $c$ be a constant satisfying $\|f_\theta \circ h_\psi \circ g_\phi(\mathbf{z}) - f_\theta \circ h_\psi \circ g_\phi(\mathbf{z}')\| \geq c\|\mathbf{z} - \mathbf{z}'\|$ for any $\mathbf{z}, \mathbf{z}'$ and assume that $\|f_\theta \circ h_\psi \circ g_\phi(\mathbf{0}) - \mathbf{0}\| \leq \varepsilon$. Any samples drawn from the space out of $\mathcal{D}_{\breve{\tilde{\mathbf{x}}}}$ can be correctly detected if $cr_2 - \varepsilon > r_1$.*

## 4 Experiments

### 4.1 Datasets, Baselines, and Implementation Details

We compare ImAD with "impute-then-detect" methods on 11 publicly available tabular datasets from various fields, including seven datasets with manually constructed missing values and four datasets with inherent missing values. In all experiments, only incomplete normal data are used in the training stage, but there are both incomplete normal and abnormal data during the inference. The statistics of all datasets are in Table 1 and a detailed description of all datasets is in Appendix J. Considering the "impute-then-detect" strategy, for data imputation, we use MissForest [Stekhoven and Bühlmann, 2012] and GAIN [Yoon et al., 2018]. For anomaly detection, we use Isolation Forest [Liu et al.,

2008], Deep SVDD [Ruff et al., 2018], NeutraL AD [Qiu et al., 2021] and DPAD [Fu et al., 2024]. The pairwise combination between the imputation and anomaly detection methods yields eight "impute-then-detect" baselines.

Table 1: Statistics of datasets. The "normal" and "abnormal" denote the number of normal and abnormal samples, respectively. "missing samples rate" means the proportion of samples with missing values and "missing entries rate" means the proportion of all missing values.

|  | dataset | field | features | instances | normal | abnormal |
|---|---|---|---|---|---|---|
| without inherent missing values | Adult | income census | 14 | 30,162 | 22,658 | 7,508 |
|  | Botnet | cybersecurity | 115 | 40,607 | 13,113 | 27,494 |
|  | KDD | cybersecurity | 121 | 494,021 | 396,743 | 97,278 |
|  | Arrhythmia | medical diagnosis | 274 | 452 | 320 | 132 |
|  | Speech | speech recognition | 400 | 3,686 | 3,625 | 61 |
|  | Segerstolpe | cell analysis | 1,000 | 702 | 329 | 372 |
|  | Usoskin | cell analysis | 25,334 | 610 | 232 | 378 |

|  | dataset | field | features | instances | missing samples rate | missing entries rate |
|---|---|---|---|---|---|---|
| with inherent missing values | Titanic | pattern recognition | 9 | 891 | 79.46% | 10.79% |
|  | MovieLens1M | recommendation system | 498 | 6,040 | 100% | 82.41% |
|  | Bladder | cell analysis | 23,341 | 2,500 | 100% | 86.93% |
|  | Seq2-Heart | cell analysis | 23,341 | 4,365 | 100% | 88.51% |

We use MLPs to construct the three modules of ImAD, Adam [Kingma and Ba, 2015] as the optimizer and set coefficient $\eta$ of entropy regularization term in Sinkhorn distance to 0.1 in all experiments. Other experimental hyper-parameters are provided in Appendix J. Sensitivity analysis of hyper-parameters is provided in Appendix I. A detailed description of distinct missing mechanisms, including MCAR, MAR, and MNAR, is provided in Appendix J. In this study, we let the missing rate mr be 0.2 or 0.5, which is consistent with the previous data imputation works [Yoon et al., 2018, Muzellec et al., 2020]. We use the AUROC (Area Under the Receiver Operating Characteristic curve) and AUPRC (Area Under the Precision-Recall curve) to evaluate the detection performance. ALL experiments were conducted on 20 Cores Intel(R) Xeon(R) Gold 6248 CPU with one NVIDIA Tesla V100 GPU, CUDA 12.0. We report the average results of five runs.

## 4.2 Experimental Results on Datasets with Manually Constructed Missing Values

Before presenting the numerical results, we show the effectiveness of the generated pseudo-abnormal samples learned for the Botnet dataset in Figure 5, where we directly let the latent space $\mathcal{Z}$ be 2-D for convenient visualization. We see that the pseudo-abnormal samples cover the region of real abnormal samples, which matches our motivation and expectation.

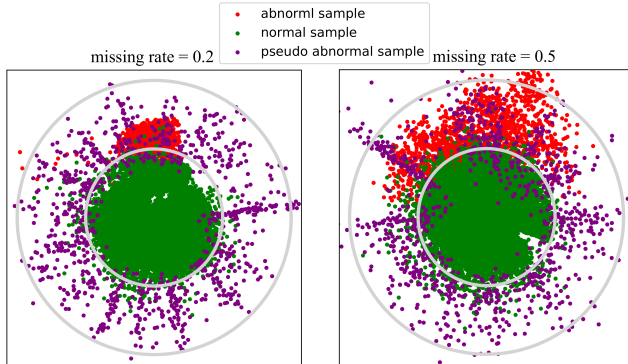

Figure 5: Two-dimensional visualization on Botnet.

The results of anomaly detection with missing data under the setting of MCAR are shown in Table 2 and more results under MCAR are provided in Appendix K. In Table 2, "Mean-Filling" denotes that the missing values are filled with feature means.

We have the following observations from the Table 2:

- The detection performance of "impute-then-detect" methods does not decrease with the increasing of missing rate from 0.2 to 0.5 in some cases (emphasized by underline), which indicates the adverse impact of imputation bias for the detection algorithm. The main reason is that a

Table 2: Detection performance in terms of AUROC and AUPRC (%, mean and std) on datasets with manually constructed missing values under MCAR. mr denotes the missing rate. The best result in each case is marked in **bold**.

| DI Methods | AD Methods | KDD | | | | Adult | | | |
|---|---|---|---|---|---|---|---|---|---|
| | | AUROC(%) | | AUPRC(%) | | AUROC(%) | | AUPRC(%) | |
| | | mr = 0.2 | mr = 0.5 | mr = 0.2 | mr = 0.5 | mr = 0.2 | mr = 0.5 | mr = 0.2 | mr = 0.5 |
| Mean-Filling | I-Forest | 74.71(1.92) | 57.44(5.71) | 81.19(1.31) | 64.37(5.95) | 59.63(0.56) | 56.01(0.84) | 57.99(1.03) | 59.25(0.72) |
| | Deep SVDD | 95.44(1.17) | 85.32(4.26) | 95.09(1.36) | 84.91(4.31) | 62.53(5.35) | 57.37(2.55) | 62.28(6.81) | 59.03(3.19) |
| | NeutraL AD | 96.63(1.05) | 87.39(1.72) | 88.69(1.36) | 88.10(2.87) | 60.37(2.50) | 54.64(1.08) | 63.20(2.86) | 57.43(1.16) |
| | DPAD | 52.55(2.41) | 53.73(2.45) | 55.21(3.03) | 53.51(2.46) | 61.93(0.17) | 59.17(0.35) | 66.72(0.10) | 62.42(0.13) |
| MissForest | I-Forest | 94.90(1.95) | **93.37**(2.15) | 93.24(2.38) | 93.21(1.92) | 60.06(1.69) | 60.73(0.69) | 57.12(2.16) | 56.80(1.27) |
| | Deep SVDD | 93.58(2.46) | 91.84(5.91) | 85.77(2.95) | 88.79(1.29) | 62.33(4.86) | 61.21(2.24) | 55.31(2.91) | 55.45(1.72) |
| | NeutraL AD | 94.00(1.72) | 92.68(2.44) | 93.87(1.57) | **94.88**(2.86) | 58.79(1.88) | 55.12(3.41) | 50.07(6.50) | 52.27(3.61) |
| | DPAD | 70.65(4.61) | 60.80(1.08) | 76.37(2.58) | 62.07(0.08) | 64.39(0.15) | 63.82(0.23) | 68.68(0.31) | 64.80(0.13) |
| GAIN | I-Forest | 82.78(3.80) | 79.94(0.39) | 90.33(1.58) | 89.52(1.07) | 59.53(0.91) | 61.18(1.61) | 57.05(1.02) | 56.87(1.09) |
| | Deep SVDD | 88.68(4.87) | 88.44(5.54) | 88.36(3.42) | 85.45(5.67) | 58.65(3.44) | 56.44(2.13) | 57.61(4.24) | 59.55(2.34) |
| | NeutraL AD | 90.48(3.24) | 84.10(0.91) | 84.61(1.30) | 84.08(1.71) | 55.04(1.81) | 56.44(2.13) | 53.00(6.80) | 59.06(3.97) |
| | DPAD | 70.34(8.20) | 90.80(0.09) | 72.29(10.59) | 94.49(0.04) | 62.10(0.85) | 62.60(0.18) | 68.39(0.42) | 68.48(0.21) |
| ImAD (Ours) | | **97.01**(0.33) | 90.78(1.35) | **95.96**(0.18) | 91.58(0.32) | **76.51**(2.12) | **71.19**(1.63) | **73.42**(2.08) | **71.50**(2.02) |

| DI Methods | AD Methods | Arrhythmia | | | | Speech | | | |
|---|---|---|---|---|---|---|---|---|---|
| | | AUROC(%) | | AUPRC(%) | | AUROC(%) | | AUPRC(%) | |
| | | mr = 0.2 | mr = 0.5 | mr = 0.2 | mr = 0.5 | mr = 0.2 | mr = 0.5 | mr = 0.2 | mr = 0.5 |
| Mean-Filling | I-Forest | 78.83(1.48) | 76.29(1.62) | 75.83(0.86) | 76.05(2.16) | 26.28(1.45) | 34.54(2.03) | 36.10(0.47) | 38.98(0.79) |
| | Deep SVDD | 66.22(1.89) | 62.60(4.67) | 71.76(1.81) | 66.46(3.93) | 53.90(5.43) | 52.80(1.57) | 57.52(4.22) | 53.75(2.67) |
| | NeutraL AD | 79.02(1.91) | 74.87(2.15) | 81.02(1.97) | 76.79(2.34) | 49.54(3.91) | 50.65(4.58) | 50.79(2.26) | 51.82(1.87) |
| | DPAD | 78.33(1.47) | 73.37(3.23) | 78.97(1.46) | 74.63(2.41) | 40.22(2.79) | 42.22(4.16) | 46.26(2.39) | 45.58(2.01) |
| MissForest | I-Forest | 80.72(0.62) | 81.54(0.95) | 77.91(1.85) | 77.95(0.97) | 28.58(2.95) | 29.09(1.14) | 36.83(1.06) | 37.29(0.76) |
| | Deep SVDD | 72.63(0.99) | 75.80(4.07) | 70.94(0.75) | 77.39(4.55) | 60.37(0.87) | 40.14(4.30) | 58.93(1.35) | 42.08(2.16) |
| | NeutraL AD | 47.38(4.81) | 44.30(2.11) | 50.87(3.53) | 50.12(2.52) | 56.51(4.87) | 54.11(3.77) | 55.44(4.36) | 52.26(3.97) |
| | DPAD | 80.79(1.11) | **82.64**(1.27) | 80.35(1.76) | 83.30(1.42) | 44.81(5.10) | 43.32(3.71) | 48.57(3.26) | 47.63(3.88) |
| GAIN | I-Forest | 77.19(0.81) | 76.29(1.35) | 76.40(1.86) | 76.29(1.35) | 29.33(0.59) | 29.23(1.35) | 39.92(0.21) | 40.04(0.63) |
| | Deep SVDD | 57.14(5.41) | 48.86(2.35) | 59.35(2.58) | 54.03(2.45) | 54.95(1.79) | 46.54(2.10) | 54.38(0.96) | 47.54(1.75) |
| | NeutraL AD | 37.96(5.09) | 33.98(4.12) | 42.57(2.56) | 42.35(1.96) | 56.80(4.89) | 57.24(5.51) | 54.76(4.58) | 55.05(5.58) |
| | DPAD | 79.07(1.29) | 80.11(4.13) | 76.48(0.97) | 79.67(4.77) | 41.91(3.78) | 44.95(2.16) | 46.46(2.79) | 49.69(2.18) |
| ImAD (Ours) | | **82.24**(1.76) | 81.76(1.19) | **83.74**(1.85) | **83.37**(1.36) | **61.94**(2.77) | **58.66**(1.40) | **60.43**(3.33) | **58.13**(1.48) |

    lower missing rate implies a simpler imputation task, leading to a more pronounced imputation bias from normal data, which makes the abnormal data more "normal", thereby increasing the difficulty of detection for such two-stage methods.

- The "impute-then-detect" methods with "MissForest" (simple and shallow imputation algorithms) achieve better detection performance than those with "GAIN" (generative and deep imputation model) in most cases, suggesting that a sophisticated imputation module may not contribute positively to subsequent anomaly detection because the identical distribution assumption does not hold here. The outstanding recovery ability leads to a pronounced imputation bias and further affects the detection task.
- Compared with all baselines, ImAD achieves better detection performance in almost all cases. Besides, different from the "impute-then-detect" methods, the performance of ImAD increases with the changes of missing rate from $0.5$ to $0.2$ in all cases. This indicates that the imputation module of ImAD generalizes well on incomplete abnormal data and the generated pseudo-abnormal samples can alleviate the bias.

### 4.3 Experimental Results on Datasets with Inherent Missing Values

We report experimental results on the four datasets with inherent missing values in Table 3, where the naive imputation methods "Zero-Filling" and 'Mean-Filling' are also considered. Observing Table 3, we notice that the naive imputation methods are insufficient for subsequent detection tasks when facing high missing rates and the imputation bias impacts the detection accuracy of "impute-then-detect" methods. Our ImAD outperforms all baselines in all cases. It indicates that our proposed method is practical and effective for real-world anomaly detection with missing data.

### 4.4 Impact of Different Missing Mechanisms

Given a real dataset, the missing mechanism is usually unknown and difficult to estimate. It is expected that when the missing mechanism $\tilde{M}$ in generating (incomplete) pseudo-normal samples is closer to the missing mechanism $M$ in the real data, the performance of ImAD should be better. In this section, we analyze the impact of different $\tilde{M}$ on the detection performance of ImAD. Note that

Table 3: Detection accuracy (AUROC and AUPRC (%, mean and std)) on four real-world datasets with inherent missing values. The best result in each case is marked in **bold**.

| DI Methods | AD Methods | Titanic | | MovieLens1M | | Bladder | | Seq2-Heart | |
|---|---|---|---|---|---|---|---|---|---|
| | | AUROC(%) | AUPRC(%) | AUROC(%) | AUPRC(%) | AUROC(%) | AUPRC(%) | AUROC(%) | AUPRC(%) |
| Zero-Filling | I-Forest | 77.44(0.29) | 77.74(0.17) | 35.53(0.43) | 40.87(0.32) | 32.22(0.72) | 39.23(0.41) | 48.78(0.63) | 46.25(0.54) |
| | Deep SVDD | 54.03(1.88) | 53.49(3.21) | 41.38(0.98) | 43.44(0.58) | 68.40(2.37) | 76.32(2.44) | 71.12(0.90) | 65.19(0.95) |
| | NeutraL AD | 49.94(0.58) | 47.10(1.31) | 39.29(0.79) | 44.05(0.72) | 63.29(0.39) | 64.86(0.84) | 82.84(3.37) | 78.76(2.17) |
| | DPAD | 79.50(0.91) | 79.07(1.30) | 44.10(3.46) | 46.12(2.29) | 99.99(0.00) | 99.99(0.00) | 95.31(0.32) | 93.38(0.69) |
| Mean-Filling | I-Forest | 79.60(0.65) | 78.64(0.94) | 36.30(0.75) | 41.47(0.46) | 44.06(3.12) | 46.63(3.88) | 54.69(2.32) | 51.98(2.07) |
| | Deep SVDD | 53.87(0.09) | 52.41(0.36) | 48.18(2.06) | 46.64(0.93) | 81.97(3.44) | 78.89(3.78) | 75.16(1.05) | 71.12(0.65) |
| | NeutraL AD | 66.15(2.00) | 63.38(2.77) | 38.34(0.69) | 42.15(0.46) | 99.15(0.39) | 99.38(2.64) | 89.87(8.36) | 86.57(8.50) |
| | DPAD | 67.02(0.66) | 69.85(0.88) | 47.74(4.08) | 48.52(2.89) | 97.52(1.43) | 97.88(1.15) | 77.93(10.20) | 76.37(8.95) |
| MissForest | I-Forest | 79.72(0.28) | 78.50(0.33) | 36.34(1.04) | 41.45(0.25) | 44.53(2.84) | 46.84(1.94) | 64.58(3.98) | 58.56(3.10) |
| | Deep SVDD | 60.46(8.59) | 60.78(3.73) | 56.04(0.49) | 53.79(0.41) | 68.40(0.81) | 92.30(0.45) | 94.29(0.42) | 92.30(0.45) |
| | NeutraL AD | 54.63(4.42) | 52.13(3.16) | 57.14(1.18) | 55.07(1.72) | 66.41(4.66) | 68.01(2.52) | 91.80(1.18) | 90.87(1.16) |
| | DPAD | 68.18(1.45) | 70.01(0.31) | 47.50(4.40) | 48.49(3.04) | 96.96(1.30) | 97.29(1.07) | 78.03(5.20) | 74.67(5.33) |
| GAIN | I-Forest | 79.46(0.79) | 78.69(0.96) | 64.84(1.04) | 62.63(1.09) | 45.77(2.39) | 47.62(2.19) | 64.62(3.31) | 58.82(2.32) |
| | Deep SVDD | 70.59(4.54) | 66.43(4.17) | 58.99(1.81) | 56.68(2.11) | 95.43(1.18) | 96.78(0.47) | 93.93(0.37) | 91.54(0.67) |
| | NeutraL AD | 53.71(2.74) | 51.55(2.25) | 50.72(2.88) | 51.47(2.38) | 65.30(3.98) | 65.68(4.69) | 91.48(0.76) | 90.79(1.36) |
| | DPAD | 78.12(0.97) | 77.41(1.04) | 59.98(1.86) | 58.98(1.90) | 96.89(1.85) | 97.25(1.56) | 74.99(2.76) | 73.16(1.93) |
| ImAD (Ours) | | **82.09**(0.99) | **81.39**(0.84) | **66.32**(1.36) | **65.34**(1.35) | **100**(0.00) | **100**(0.00) | **96.62**(0.11) | **96.40**(0.19) |

for the synthetic incomplete data, we accurately know the missing mechanism. The experimental results are reported in Table 4. On real incomplete data, our method is robust to the setting of missing mechanism $\tilde{M}$ and has better overall performance when $\tilde{M}$ is MCAR. Therefore, based on Occam's Razor principle and the empirical results, we recommend using MCAR as the missing mechanism for the generated pseudo-abnormal samples when the real missing mechanism is unknown. On the other hand, as shown in Table 4, on synthetic incomplete data, detection performance degrades when $\tilde{M}$ is different from $M$.

Table 4: Performance comparison of different missing mechanisms $\tilde{M}$.

| Dataset | Missing Mechanism M of Normal Data | Missing Mechanism of Pseudo-Abnormal Samples $\tilde{M}$ | | | | | |
|---|---|---|---|---|---|---|---|
| | | MCAR | | MAR | | MNAR | |
| | | AUROC(%) | AUPRC(%) | AUROC(%) | AUPRC(%) | AUROC(%) | AUPRC(%) |
| Titanic | Unknown | 82.09 | 81.39 | 79.06 | 77.08 | 80.50 | 79.17 |
| MovieLens1M | Unknown | 66.32 | 65.34 | 63.14 | 63.39 | 61.44 | 60.91 |
| Bladder | Unknown | 100.00 | 100.00 | 99.95 | 99.95 | 100.00 | 100.00 |
| Seq2_Heart | Unknown | 96.62 | 96.40 | 96.79 | 96.60 | 95.56 | 94.41 |
| Adult | MCAR | 71.19 | 71.50 | 64.11 | 66.44 | 67.28 | 66.72 |
| Adult | MAR | 65.66 | 67.23 | 74.61 | 70.74 | 71.14 | 69.69 |
| Adult | MNAR | 70.69 | 69.17 | 68.35 | 68.78 | 71.60 | 68.97 |

## 4.5 More Experimental Results

The appendices contain the following additional results: I. Performance gain from pseudo-abnormal samples (Appendix G); II. Influence of the constrained radii $r_1, r_2$ (Appendix H); III. Sensitivity analysis of hyperparameters (Appendix I); IV. Impact of different missing rates for training and test set(Appendix K); V. Results of MAR and MNAR (Appendix K).

## 5 Conclusion

This paper proposed ImAD, the first end-to-end unsupervised anomaly detection method on incomplete data. ImAD integrates data imputation with anomaly detection into a unified optimization objective and automatically generates pseudo-abnormal samples to alleviate the imputation bias. We theoretically proved the effectiveness of ImAD and empirically evaluated ImAD on multiple real-world datasets. The results showed that ImAD mitigates imputation bias from normal data and provides an effective solution for unsupervised anomaly detection in the presence of missing values. One limitation of this work is that we haven't considered the applications on incomplete image data and incomplete time series.

## Acknowledgments

This work was supported by the General Program of Natural Science Foundation of Guangdong Province under Grant No.2024A1515011771, the National Natural Science Foundation of China under Grant No.62376236, the Guangdong Provincial Key Laboratory of Mathematical Foundations for Artificial Intelligence (2023B1212010001), Shenzhen Science and Technology Program ZDSYS20230626091302006, Shenzhen Stability Science Program 2023, and Hetao Shenzhen-Hong Kong Science and Technology Innovation Cooperation Zone Project (No.HZQSWS-KCCYB-2024016). The authors declare that they have no known competing financial interests or personal relationships that could have appeared to influence the work reported in this paper.

## Footnotes

[13]https://github.com/jsyoon0823/GAIN

[14]https://github.com/BorisMuzellec/MissingDataOT

[15]https://pypi.org/project/missingpy/

[16]https://github.com/lukasruff/Deep-SVDD-PyTorch

[17]https://github.com/boschresearch/NeuTraL-AD

[18]https://scikit-learn.org/stable/

## References

Peter L Bartlett, Dylan J Foster, and Matus J Telgarsky. Spectrally-normalized margin bounds for neural networks. *Advances in neural information processing systems*, 30, 2017.

Barry Becker and Ronny Kohavi. Adult. UCI Machine Learning Repository, 1996. DOI: https://doi.org/10.24432/C5XW20.

Markus M Breunig, Hans-Peter Kriegel, Raymond T Ng, and Jörg Sander. Lof: identifying density-based local outliers. In *Proceedings of the 2000 ACM SIGMOD international conference on Management of data*, pages 93–104, 2000.

Jinyu Cai and Jicong Fan. Perturbation learning based anomaly detection. *Advances in Neural Information Processing Systems*, 35, 2022.

Emmanuel Candes and Benjamin Recht. Exact matrix completion via convex optimization. *Communications of the ACM*, 55(6):111–119, 2012.

Zhenpeng Chen, Jie M Zhang, Max Hort, Mark Harman, and Federica Sarro. Fairness testing: A comprehensive survey and analysis of trends. *ACM Transactions on Software Engineering and Methodology*, 2023.

Marco Cuturi. Sinkhorn distances: Lightspeed computation of optimal transport. *Advances in neural information processing systems*, 26, 2013.

Arthur P Dempster, Nan M Laird, and Donald B Rubin. Maximum likelihood from incomplete data via the em algorithm. *Journal of the Royal Statistical Society: Series B (Methodological)*, 39(1): 1–22, 1977.

Botao Fa, Ting Wei, Yuan Zhou, Luke Johnston, Xin Yuan, Yanran Ma, Yue Zhang, and Zhangsheng Yu. Gapclust is a light-weight approach distinguishing rare cells from voluminous single cell expression profiles. *Nature Communications*, 12(1):4197, 2021.

Jicong Fan and Tommy Chow. Deep learning based matrix completion. *Neurocomputing*, 266: 540–549, 2017.

Jicong Fan and Tommy WS Chow. Non-linear matrix completion. *Pattern Recognition*, 77:378–394, 2018.

Jicong Fan, Lijun Ding, Yudong Chen, and Madeleine Udell. Factor group-sparse regularization for efficient low-rank matrix recovery. *Advances in neural information processing Systems*, 32, 2019.

Jicong Fan, Yuqian Zhang, and Madeleine Udell. Polynomial matrix completion for missing data imputation and transductive learning. In *Proceedings of the AAAI Conference on Artificial Intelligence*, volume 34, pages 3842–3849, 2020.

Jicong Fan, Tommy W. S. Chow, and S. Joe Qin. Kernel-based statistical process monitoring and fault detection in the presence of missing data. *IEEE Transactions on Industrial Informatics*, 18 (7):4477–4487, 2022. doi: 10.1109/TII.2021.3119377.

Jicong Fan, Rui Chen, Zhao Zhang, and Chris H.Q. Ding. Neuron-enhanced autoencoder matrix completion: Theory and practice. In *The Twelfth International Conference on Learning Representations*, 2024.

Dazhi Fu, Zhao Zhang, and Jicong Fan. Dense projection for anomaly detection. In *Proceedings of the AAAI Conference on Artificial Intelligence*, volume 38, pages 8398–8408, 2024.

Lovedeep Gondara and Ke Wang. Mida: Multiple imputation using denoising autoencoders. In *Advances in Knowledge Discovery and Data Mining: 22nd Pacific-Asia Conference, PAKDD 2018, Melbourne, VIC, Australia, June 3-6, 2018, Proceedings, Part III 22*, pages 260–272. Springer, 2018.

Ian Goodfellow, Jean Pouget-Abadie, Mehdi Mirza, Bing Xu, David Warde-Farley, Sherjil Ozair, Aaron Courville, and Yoshua Bengio. Generative adversarial nets. *Advances in neural information processing systems*, 27, 2014.

Soyeon Caren Han, Taejun Lim, Siqu Long, Bernd Burgstaller, and Josiah Poon. Glocal-k: Global and local kernels for recommender systems. In *Proceedings of the 30th ACM International Conference on Information & Knowledge Management*, pages 3063–3067, 2021.

Xiao Han, Lu Zhang, Yongkai Wu, and Shuhan Yuan. Achieving counterfactual fairness for anomaly detection. In *Pacific-Asia Conference on Knowledge Discovery and Data Mining*, pages 55–66. Springer, 2023.

Diederik P Kingma and Jimmy Ba. Adam: A method for stochastic optimization. In *Proceedings of the International Conference on Learning Representations*, 2015.

Steven Cheng-Xian Li, Bo Jiang, and Benjamin Marlin. Misgan: Learning from incomplete data with generative adversarial networks. *arXiv preprint arXiv:1902.09599*, 2019.

M. Lichman. Uci machine learning repository, 2013. URL `http://archive.ics.uci.edu/ml`.

Fei Tony Liu, Kai Ming Ting, and Zhi-Hua Zhou. Isolation forest. In *2008 eighth ieee international conference on data mining*, pages 413–422. IEEE, 2008.

Imke Mayer, Aude Sportisse, Julie Josse, Nicholas Tierney, and Nathalie Vialaneix. R-miss-tastic: a unified platform for missing values methods and workflows. *arXiv preprint arXiv:1908.04822*, 2019.

Rahul Mazumder, Trevor Hastie, and Robert Tibshirani. Spectral regularization algorithms for learning large incomplete matrices. *The Journal of Machine Learning Research*, 11:2287–2322, 2010.

Yair Meidan, Michael Bohadana, Yael Mathov, Yisroel Mirsky, Dominik Breitenbacher, Asaf, and Asaf Shabtai. detection_of_iot_botnet_attacks_n_baiot. UCI Machine Learning Repository, 2018. DOI: https://doi.org/10.24432/C5RC8J.

Boris Muzellec, Julie Josse, Claire Boyer, and Marco Cuturi. Missing data imputation using optimal transport. In *International Conference on Machine Learning*, pages 7130–7140. PMLR, 2020.

Tomáš Pevný. Loda: Lightweight on-line detector of anomalies. *Machine Learning*, 102:275–304, 2016.

Therese D Pigott. A review of methods for missing data. *Educational research and evaluation*, 7(4): 353–383, 2001.

Chen Qiu, Timo Pfrommer, Marius Kloft, Stephan Mandt, and Maja Rudolph. Neural transformation learning for deep anomaly detection beyond images. In *International Conference on Machine Learning*, pages 8703–8714. PMLR, 2021.

Shebuti Rayana. Odds library, 2016. URL `https://odds.cs.stonybrook.edu`.

Patrick Royston and Ian R White. Multiple imputation by chained equations (mice): implementation in stata. *Journal of statistical software*, 45:1–20, 2011.

Donald B Rubin. Inference and missing data. *Biometrika*, 63(3):581–592, 1976.

Lukas Ruff, Robert Vandermeulen, Nico Goernitz, Lucas Deecke, Shoaib Ahmed Siddiqui, Alexander Binder, Emmanuel Müller, and Marius Kloft. Deep one-class classification. In *International conference on machine learning*, pages 4393–4402. PMLR, 2018.

Kisan Sarda, Amol Yerudkar, and Carmen Del Vecchio. Unsupervised anomaly detection for multivariate incomplete data using gan-based data imputation: A comparative study. In *2023 31st Mediterranean Conference on Control and Automation (MED)*, pages 55–60. IEEE, 2023.

Nicholas Schaum, Jim Karkanias, Norma F Neff, Andrew P May, Stephen R Quake, Tony Wyss-Coray, Spyros Darmanis, Joshua Batson, Olga Botvinnik, Michelle B Chen, et al. Single-cell transcriptomics of 20 mouse organs creates a tabula muris: The tabula muris consortium. *Nature*, 562(7727):367, 2018.

Gabriel L Schlomer, Sheri Bauman, and Noel A Card. Best practices for missing data management in counseling psychology. *Journal of Counseling psychology*, 57(1):1, 2010.

Bernhard Schölkopf, John C Platt, John Shawe-Taylor, Alex J Smola, and Robert C Williamson. Estimating the support of a high-dimensional distribution. *Neural computation*, 13(7):1443–1471, 2001.

Åsa Segerstolpe, Athanasia Palasantza, Pernilla Eliasson, Eva-Marie Andersson, Anne-Christine Andréasson, Xiaoyan Sun, Simone Picelli, Alan Sabirsh, Maryam Clausen, Magnus K Bjursell, et al. Single-cell transcriptome profiling of human pancreatic islets in health and type 2 diabetes. *Cell metabolism*, 24(4):593–607, 2016.

Guy Shani and Asela Gunawardana. Evaluating recommendation systems. *Recommender systems handbook*, pages 257–297, 2011.

Jascha Sohl-Dickstein, Eric Weiss, Niru Maheswaranathan, and Surya Ganguli. Deep unsupervised learning using nonequilibrium thermodynamics. In *International conference on machine learning*, pages 2256–2265. PMLR, 2015.

Daniel J Stekhoven and Peter Bühlmann. Missforest—non-parametric missing value imputation for mixed-type data. *Bioinformatics*, 28(1):112–118, 2012.

Yusuke Tashiro, Jiaming Song, Yang Song, and Stefano Ermon. Csdi: Conditional score-based diffusion models for probabilistic time series imputation. *Advances in Neural Information Processing Systems*, 34:24804–24816, 2021.

Dmitry Usoskin, Alessandro Furlan, Saiful Islam, Hind Abdo, Peter Lönnerberg, Daohua Lou, Jens Hjerling-Leffler, Jesper Haeggström, Olga Kharchenko, Peter V Kharchenko, et al. Unbiased classification of sensory neuron types by large-scale single-cell rna sequencing. *Nature neuroscience*, 18(1):145–153, 2015.

Feng Xiao, Jianfeng Zhou, Kunpeng Han, Haoyuan Hu, and Jicong Fan. Unsupervised anomaly detection using inverse generative adversarial networks. *Information Sciences*, 689:121435, 2025. ISSN 0020-0255. doi: https://doi.org/10.1016/j.ins.2024.121435.

Zhihai Yang and Zhongmin Cai. Detecting abnormal profiles in collaborative filtering recommender systems. *Journal of Intelligent Information Systems*, 48:499–518, 2017.

Jinsung Yoon, James Jordon, and Mihaela Schaar. Gain: Missing data imputation using generative adversarial nets. In *International conference on machine learning*, pages 5689–5698. PMLR, 2018.

Tadesse Zemicheal and Thomas G. Dietterich. Anomaly detection in the presence of missing values for weather data quality control. In *Proceedings of the 2nd ACM SIGCAS Conference on Computing and Sustainable Societies*, COMPASS '19, page 65–73, New York, NY, USA, 2019. Association for Computing Machinery. ISBN 9781450367141. doi: 10.1145/3314344.3332490. URL `https://doi.org/10.1145/3314344.3332490`.

Lihua Zhang and Shihua Zhang. Comparison of computational methods for imputing single-cell rna-sequencing data. *IEEE/ACM transactions on computational biology and bioinformatics*, 17(2):376–389, 2018.

Yunhe Zhang, Yan Sun, Jinyu Cai, and Jicong Fan. Deep orthogonal hypersphere compression for anomaly detection. In *The Twelfth International Conference on Learning Representations*, 2024. URL `https://openreview.net/forum?id=cJs4oE4m9Q`.

Bo Zong, Qi Song, Martin Renqiang Min, Wei Cheng, Cristian Lumezanu, Daeki Cho, and Haifeng Chen. Deep autoencoding gaussian mixture model for unsupervised anomaly detection. In *International conference on learning representations*, 2018.

# A   Analysis for Sampling in Latent Space

To project the normal data onto target distribution and generate pseudo-abnormal data, our proposed method involves sampling from two latent distributions $\mathcal{D}_{\mathbf{z}}, \mathcal{D}_{\tilde{\mathbf{z}}} \sim \mathcal{N}(\mathbf{0}, \sigma^2 \mathbf{I}_d)$. In this section, we provide a lower bound for the constrained sampling radius $r$, given a sampling probability $p$ that provides a probabilistic guarantee, namely, **to obtain $N$ points from a truncated Gaussian, we need to sample $N/p$ times from a Gaussian distribution**. Subsequently, we perform sampling within the truncated Gaussian distribution with a constrained radius $r$.

For target distribution $\mathcal{D}_{\mathbf{z}}$, we expect that it is compact and can be easily sampled, in which the compactness is to ensure a clear and reliable decision boundary between normal and abnormal data. Therefore, we select truncated Gaussian from $\mathcal{N}(\mathbf{0}, \sigma^2 \mathbf{I}_d)$ as target distribution $\mathcal{D}_{\mathbf{z}}$ and bound $\mathcal{D}_{\mathbf{z}}$ in a $d$-dimensional radius $r$ hyperball centering at origin. For radius $r$, we have the following proposition.

**Proposition A.1.** *Let $F_d$ denote the cumulative distribution function (CDF) of the chi-square distribution $\chi^2(d)$. For a given probability $0 < p < 1$, when $r \geq \sigma \sqrt{F_d^{-1}(p)}$, the sampling probability in $\mathcal{D}_{\mathbf{z}}$ satisfies $P(\|\mathbf{z}\|^2 < r^2) \geq p$ where $\mathbf{z} = [z_1, z_2, \cdots, z_d]$ and $z_1, \ldots, z_d \overset{i.i.d.}{\sim} \mathcal{N}(0, \sigma^2)$.*

*Proof.* We have

$$z_1, \ldots, z_d \overset{\text{i.i.d.}}{\sim} \mathcal{N}(0, \sigma^2) \implies \frac{z_1}{\sigma}, \ldots, \frac{z_d}{\sigma} \overset{\text{i.i.d.}}{\sim} \mathcal{N}(0, 1) \implies \frac{\sum_{i=1}^d z_i^2}{\sigma^2} \sim \chi^2(d). \tag{15}$$

Let $Y = \frac{\sum_{i=1}^d z_i^2}{\sigma^2}$, we get

$$
\begin{aligned}
&P\left(Y < F_d^{-1}(p)\right) = p \\
\implies\ &P\left(\frac{\sum_{i=1}^d z_i^2}{\sigma^2} < F_d^{-1}(p)\right) = p \\
\implies\ &P\left(\sum_{i=1}^d z_i^2 < \sigma^2 \cdot F_d^{-1}(p)\right) = p \\
\implies\ &P\left(\|\mathbf{z}\|^2 < \left(\sigma\sqrt{F_d^{-1}(p)}\right)^2\right) = p.
\end{aligned}
\tag{16}
$$

Therefore, $r \geq \sigma \sqrt{F_d^{-1}(p)} \implies P\left(\|\mathbf{z}\|^2 < r^2\right) \geq p$. $\qquad\square$

According to the analysis in Section 3.2, we select truncated Gaussian from $\mathcal{N}(\mathbf{0}, \tilde{\sigma}^2 \mathbf{I}_d)$ as target distribution $\mathcal{D}_{\tilde{\mathbf{z}}}$ and bound $\mathcal{D}_{\tilde{\mathbf{z}}}$ between two $d$-dimensional hyperspheres with radii $r_1, r_2$ respectively, centering at origin, where $r_2 > r_1$. For radius $r_1, r_2$, we have the following proposition.

**Proposition A.2.** *Let $F_d$ denote the cumulative distribution function (CDF) of the chi-square distribution $\chi^2(d)$. For a given probability $0 < p < 1$, when $r_1 \leq \tilde{\sigma}\sqrt{F_d^{-1}(p_1)}, r_2 \geq \tilde{\sigma}\sqrt{F_d^{-1}(p_2)}$ and satisfies $p = p_2 - p_1$, the sampling probability in $\mathcal{D}_{\tilde{\mathbf{z}}}$ satisfies $P(r_1^2 < \|\mathbf{z}\|^2 < r_2^2) \geq p$ where $\mathbf{z} = [z_1, z_2, \cdots, z_d]$ and $z_1, \ldots, z_d \overset{i.i.d.}{\sim} \mathcal{N}(0, \tilde{\sigma}^2)$.*

*Proof.* According the proof for Proposition A.1, we have

$$r \geq \tilde{\sigma}\sqrt{F_d^{-1}(p)} \implies P(\|\mathbf{z}\|^2 < r^2) \geq p. \tag{17}$$

Therefore,

$$
\begin{aligned}
r_1 \leq \tilde{\sigma}\sqrt{F_d^{-1}(p_1)} &\implies P(\|\mathbf{z}\|^2 < r_1^2) \leq p_1, \text{and} \\
r_2 \geq \tilde{\sigma}\sqrt{F_d^{-1}(p_2)} &\implies P(\|\mathbf{z}\|^2 < r_2^2) \geq p_2.
\end{aligned}
\tag{18}
$$

Therefore, we get $P\left(\|\mathbf{z}\|^2 < r_2^2\right) - P(\|\mathbf{z}\|^2 < r_1^2) = P\left(r_1^2 < \|\mathbf{z}\|^2 < r_2^2\right) \geq p_2 - p_1 = p$. $\qquad\square$

As shown in Figure 3, we set radius $r_1$ of $\mathcal{D}_{\tilde{\mathbf{z}}}$ equals to radius $r$ of $\mathcal{D}_{\mathbf{z}}$. Also, we maintain the settings $r_1 = r$ in all experiments to make the introduced pseudo-abnormal samples are not far from the normal data.

## B    Proof for Theorem 3.2

*Proof.* Recall that $g_\phi$ was defined as

$$g_\phi(\mathbf{z}) = \sigma_L(\mathbf{W}_L^g(\cdots\sigma_2(\mathbf{W}_2^g(\sigma_1(\mathbf{W}_1^g\mathbf{z})))\cdots)). \tag{19}$$

Then for any $\mathbf{z}, \tilde{\mathbf{z}} \in \mathbb{R}^d$, we have

$$
\begin{aligned}
&\|g_\phi(\mathbf{z}) - g_\phi(\tilde{\mathbf{z}})\| \\
=&\|\sigma_L(\mathbf{W}_L^g(\cdots\sigma_2(\mathbf{W}_2^g(\sigma_1(\mathbf{W}_1^g\mathbf{z})))\cdots)) - \sigma_L(\mathbf{W}_L^g(\cdots\sigma_2(\mathbf{W}_2^g(\sigma_1(\mathbf{W}_1^g\tilde{\mathbf{z}})))\cdots))\| \\
\leq&\rho\|\mathbf{W}_L^g(\cdots\sigma_2(\mathbf{W}_2^g(\sigma_1(\mathbf{W}_1^g\mathbf{z})))\cdots) - \mathbf{W}_L^g(\cdots\sigma_2(\mathbf{W}_2^g(\sigma_1(\mathbf{W}_1^g\tilde{\mathbf{z}})))\cdots)\| \\
\leq&\rho\|\mathbf{W}_L^g\|_\sigma\|\sigma_{L-1}(\cdots\sigma_2(\mathbf{W}_2^g(\sigma_1(\mathbf{W}_1^g\mathbf{z})))\cdots) - \sigma_{L-1}(\cdots\sigma_2(\mathbf{W}_2^g(\sigma_1(\mathbf{W}_1^g\tilde{\mathbf{z}})))\cdots)\| \\
\leq&\rho^L\left(\prod_{l=1}^L\|\mathbf{W}_l^g\|_\sigma\right)\|\mathbf{z} - \tilde{\mathbf{z}}\| \\
\leq&\rho^L\alpha_g^L\|\mathbf{z} - \tilde{\mathbf{z}}\|.
\end{aligned} \tag{20}
$$

This finished the proof for part (a) of the theorem. The proof for part (b) is similar and omitted here for simplicity. $\square$

## C    Proof for Theorem 3.3

We define the following model class

$$\mathcal{F} = \{\pi \circ f_\theta \circ h_\psi : \mathbb{R}^d \to \mathbb{R}\} \tag{21}$$

where $f_\theta$ and $h_\psi$ satisfy the Assumption 3.1 and $\pi$ is the sum of squares of the outputs of $f_\theta \circ h_\psi$, corresponding to the definition of the anomaly score, meaning $s(\mathbf{x})^2 = \pi(f_\theta(h_\psi(\mathbf{x})))$. The following lemma (proved by Appendix E) provides the covering number bound of $\mathcal{F}$.

**Lemma C.1.** *Under Assumption 3.1, for any $\epsilon > 0$, it holds that*

$$\ln\mathcal{N}\left(\epsilon, \mathcal{F}_{\check{\mathbf{X}}}, \|\cdot\|_F\right) \leq \frac{\|\check{\mathbf{X}}\|_F^2 \ln(\bar{d}^2)\rho^{4L-2}(2\vartheta)^2 d\kappa^2\zeta^2}{\epsilon^2}$$

*where $\kappa^2 = \alpha_f^{2L}\alpha_h^{2L}$ and $\zeta^2 = \left(1 + L\left(\frac{b_f}{\alpha_f}\right)^{2/3} + L\left(\frac{b_h}{\alpha_h}\right)^{2/3}\right)^3$.*

Suppose the loss function is $\mu$-Lipschitz, it follows from Lemma C.1 that

$$
\begin{aligned}
\ln\mathcal{N}\left(\epsilon, \ell \circ \mathcal{F}_{\check{\mathbf{X}}}, \|\cdot\|_F\right) &\leq \ln\mathcal{N}\left(\frac{\epsilon}{\mu}, \mathcal{F}_{\check{\mathbf{X}}}, \|\cdot\|_F\right) \\
&\leq \frac{\mu^2\|\check{\mathbf{X}}\|_F^2 \ln(2\bar{d}^2)\rho^{4L-2}(2\vartheta)^2 d\kappa^2\zeta^2}{\epsilon^2}
\end{aligned} \tag{22}
$$

With the covering number, we can bound the Rademacher complexity by the following Dudley entropy integral:

**Lemma C.2** (Lemma A.5 of [Bartlett et al., 2017], reformulated). *Let $\mathcal{F}_\gamma := \ell \circ \mathcal{F}_{\check{\mathbf{X}}}$ be a real-valued function class taking values in $[0, \gamma]$, and assume that $\mathbf{0} \in \mathcal{F}_\gamma$. Then*

$$\mathcal{R}_{\check{\mathbf{X}}}(\mathcal{F}_\gamma) \leq \inf_{\alpha>0}\left(\frac{4\alpha\gamma}{\sqrt{n}} + \frac{12}{n}\int_{\gamma\alpha}^{\gamma\sqrt{n}}\sqrt{\ln\mathcal{N}\left(\epsilon, \mathcal{F}_\gamma, \|\cdot\|\right)}\, d\epsilon\right).$$

Combining (22) and Lemma C.2, and letting $R^2 := \mu^2 \|\breve{\mathbf{X}}\|_F^2 \ln(2\bar{d}^2)\rho^{4L}(2\vartheta)^2 d\kappa^2\zeta^2$, we obtain

$$
\begin{aligned}
\mathcal{R}_{\mathcal{G}}(\mathcal{F}_\gamma) &\leq \inf_{\alpha>0} \left( \frac{4\alpha\gamma}{\sqrt{n}} + \frac{12}{n} \int_{\gamma\alpha}^{\gamma\sqrt{n}} \frac{R}{\epsilon} d\epsilon \right) \\
&= \inf_{\alpha>0} \left( \frac{4\alpha\gamma}{\sqrt{n}} + \frac{12R}{n} \ln\left( \frac{\sqrt{n}}{\alpha} \right) \right) \\
&\leq \frac{4\gamma}{n} + \frac{12R\ln n}{n}
\end{aligned}
\tag{23}
$$

where we have chosen $\alpha = \frac{1}{\sqrt{n}}$.

The following lemma is the classical generalization error bound based on the Rademacher complexity.

**Lemma C.3.** *Given hypothesis function space $\mathcal{F}$ mapping $\mathbf{x} \in \mathcal{X}$ to $\mathbb{R}^d$ and $\gamma > 0$, define $\mathcal{F}_\gamma := \{(\mathbf{x}, y) \mapsto l_\gamma(f(\mathbf{x}), y) : f \in \mathcal{F}\}$, where $l_\gamma(\hat{y}, y) \leq \gamma$. Then, with probability at least $1 - \delta$ over a sample $\mathbf{X}$ of size $n$, every $f \in \mathcal{F}$ satisfies $L_\gamma(f) \leq \hat{L}_\gamma(f) + 2\mathcal{R}_{\mathbf{X}}(\mathcal{F}_\gamma) + 3\gamma\sqrt{\frac{\ln(2/\delta)}{2n}}$.*

Now using Lemma C.3 and inequality (23), we have

$$
L_\gamma(f) \leq \hat{L}_\gamma(f) + \frac{8\gamma + 24R\ln n}{n} + 3\gamma\sqrt{\frac{\ln(2/\delta)}{2n}}.
\tag{24}
$$

For Theorem 3.3(a), the loss function is $\ell(\hat{y}, y) = |\hat{y} - y| = \hat{y}$, where $y \equiv 0$ and $\hat{y} = s(\breve{\mathbf{x}})^2$ due to the definition of the anomaly score. This also means that the Lipschitz constant $\mu$ of $\ell$ is 1. We assume that the squared anomaly scores on the normal training data are upper bounded by $\gamma$. Let $\hat{L}_\gamma(f) = \frac{1}{n} \sum_{i=1}^n s(\breve{\mathbf{x}}_i)^2$ and $L_\gamma(f) = \mathbb{E}_{\mathcal{D}_{\breve{\mathbf{x}}}}[s(\mathbf{x})^2]$. It follows from (24) that

$$
\mathbb{E}_{\mathcal{D}_{\breve{\mathbf{x}}}}[s(\breve{\mathbf{x}})^2] \leq \frac{1}{n} \sum_{i=1}^n s(\breve{\mathbf{x}}_i)^2 + \frac{8\gamma + 24R\ln n}{n} + 3\gamma\sqrt{\frac{\ln(2/\delta)}{2n}}.
\tag{25}
$$

Let $r_1$ be the threshold determined by the training data to judge whether a sample is anomalous or not. We let

$$
\frac{1}{n} \sum_{i=1}^n s(\breve{\mathbf{x}}_i)^2 + \frac{8\gamma + 24R\ln n}{n} + 3\gamma\sqrt{\frac{\ln(2/\delta)}{2n}} = r_1^2
\tag{26}
$$

and solve for $\delta$:

$$
\delta = 2\exp\left( -\frac{2n\left(r_1^2 - \bar{s}^2 - \frac{8\gamma + 24R\ln n}{n}\right)^2}{9\gamma^2} \right)
\tag{27}
$$

where $\bar{s}^2 = \frac{1}{n} \sum_{i=1}^n s(\breve{\mathbf{x}}_i)^2$. Then we rewrite (25) as

$$
\mathbb{E}_{\mathcal{D}_{\breve{\mathbf{x}}}}[s(\breve{\mathbf{x}})^2] \leq r_1^2,
\tag{28}
$$

which holds with probability at least $1 - \delta$. In other words,

$$
\mathbb{P}\left[\mathbb{E}_{\mathcal{D}_{\breve{\mathbf{x}}}}[s(\breve{\mathbf{x}})^2] > r_1^2\right] \leq \delta,
\tag{29}
$$

which implies

$$
\mathbb{P}\left[\mathbb{E}_{\mathcal{D}_{\breve{\mathbf{x}}}}[s(\breve{\mathbf{x}})] > r_1\right] \leq \delta,
\tag{30}
$$

because both $s(\breve{\mathbf{x}})$ and $r_1$ are nonnegative. We complete the proof for Theorem 3.3(a).

For Theorem 3.3(b), $R^2 := \mu^2 \|\breve{\mathbf{X}}\|_F^2 \ln(2\bar{d}^2)\rho^{4L}(2\vartheta)^2 d\kappa^2\zeta^2$. We consider the following loss function

$$
\ell(\breve{\mathbf{x}}) = \left| r_2^2 - s(\breve{\mathbf{x}})^2 \right|.
\tag{31}
$$

The Lipshitz constant of this loss is 1. Suppose the loss is upper bounded by $\tilde{\gamma}$. Similar to the proof for Theorem 3.3 (a), we have

$$
\mathbb{E}_{\mathcal{D}_{\breve{\mathbf{x}}}}[\ell(\breve{\mathbf{x}})] \leq \frac{1}{n} \sum_{i=1}^n \ell(\breve{\mathbf{x}}_i) + \frac{8\tilde{\gamma} + 24R\ln n}{n} + 3\tilde{\gamma}\sqrt{\frac{\ln(2/\delta)}{2n}}.
\tag{32}
$$

Let

$$\frac{1}{n}\sum_{i=1}^{n}\ell(\breve{\tilde{\mathbf{x}}}_i) + \frac{8\tilde{\gamma} + 24R\ln n}{n} + 3\tilde{\gamma}\sqrt{\frac{\ln(2/\delta)}{2n}} = \tau \tag{33}$$

and solve for $\delta$:

$$\delta = 2\exp\left(-\frac{2n\left(\tau - \bar{\varepsilon} - \frac{8\tilde{\gamma}+24R\ln n}{n}\right)^2}{9\tilde{\gamma}^2}\right) \tag{34}$$

where $\bar{\varepsilon} = \frac{1}{n}\sum_{i=1}^{n}\ell(\breve{\tilde{\mathbf{x}}}_i)$. Then we rewrite (32) as

$$\mathbb{E}_{\mathcal{D}_{\breve{\tilde{\mathbf{x}}}}}[|r_2^2 - s(\breve{\tilde{\mathbf{x}}})^2|] \leq \tau, \tag{35}$$

which holds with probability at least $1 - \delta$. In other words,

$$\mathbb{P}\left[\mathbb{E}_{\mathcal{D}_{\breve{\tilde{\mathbf{x}}}}}[|r_2^2 - s(\breve{\tilde{\mathbf{x}}})^2|] \leq \tau\right] \geq 1 - \delta. \tag{36}$$

Since $|r_2^2 - s(\breve{\tilde{\mathbf{x}}})^2| \leq \tau$ implies that $s(\breve{\tilde{\mathbf{x}}})^2 \geq r_2^2 - \tau$, letting $\tau = r_2^2 - r_1^2$, we arrive at

$$\mathbb{P}\left[\mathbb{E}_{\mathcal{D}_{\breve{\tilde{\mathbf{x}}}}}[s(\breve{\tilde{\mathbf{x}}})^2] \geq r_1^2\right] \geq 1 - \delta, \tag{37}$$

which also means

$$\mathbb{P}\left[\mathbb{E}_{\mathcal{D}_{\breve{\tilde{\mathbf{x}}}}}[s(\breve{\tilde{\mathbf{x}}})] \geq r_1\right] \geq 1 - \delta, \tag{38}$$

because both $s(\breve{\tilde{\mathbf{x}}})$ and $r_1$ are nonnegative. Renaming $\delta$ as $\tilde{\delta}$, we finish the proof. Note that in the theorem of the main paper, we have put outside the constant 4 in $R^2$. That's why the constant 24 becomes 48.

## D   Proof for Theorem 3.4

*Proof.* The anomalous samples (denoted by $\breve{\mathbf{x}}$) drawn from space out of $\mathcal{D}_{\breve{\mathbf{x}}}$ are much easier to detect. The reason is that, in the latent space, these anomalous samples (denoted by $\breve{\mathbf{z}}'$) are sufficiently far from the normal region ($\mathcal{D}_{\mathbf{z}}$). According to the definition of $\mathcal{D}_{\breve{\mathbf{x}}}$, we have $\breve{\mathbf{z}}' = f \circ h(\breve{\mathbf{x}}) = f \circ h \circ g(\breve{\mathbf{z}})$. According to the definition of the anomaly score, we need to measure $\|\breve{\mathbf{z}}' - 0\|$. We have

$$
\begin{aligned}
&\|\breve{\mathbf{z}}' - 0\| \\
=&\|f \circ h \circ g(\breve{\mathbf{z}}) - f \circ h \circ g(0) + f \circ h \circ g(0) - 0\| \\
\geq&\|f \circ h \circ g(\breve{\mathbf{z}}) - f \circ h \circ g(0)\| - \|f \circ h \circ g(0) - 0\| \\
\geq& c\|\breve{\mathbf{z}} - 0\| - \|f \circ h \circ g(0) - 0\| \\
\geq& cr_2 - \varepsilon
\end{aligned}
$$

where $c$ is some constant depending on the networks and we have assumed that $\|f \circ h \circ g(0) - 0\| \leq \varepsilon$. Now suppose that $r_2$ is sufficiently large such that $cr_2 - \varepsilon > r_1$, then $\|\breve{\mathbf{z}}' - 0\| > r_1$, meaning that $\breve{\mathbf{z}}'$ is outside the inner hypersphere and hence can be detected successfully. Nevertheless, determining an exact $c$ is still an open problem for neural networks. □

## E   Proof for Lemma C.1

*Proof.* In $\mathcal{F}$, $\pi$ can be regarded as an additional layer of the neural network, where the activation function for each element of $\mathbf{z}$ is square, the weight matrix for the output is a vector consisting of $d$ ones, and the activation function for the final output is linear. Thus, $\pi \circ f_\theta \circ g_\psi$ has $2L + 1$ layers. For the square activation function, the Lipschitz constant is $2\vartheta$. For the final output layer, the spectral norm of the weights is $\sqrt{d}$, which is equal to the $\ell_{2.1}$ norm because it is a vector.

**Lemma E.1** (Theorem 3.3 of [Bartlett et al., 2017])**.** *Let fixed nonlinearities $(\sigma_1, \ldots, \sigma_L)$ and reference matrices $(M_1, \ldots, M_L)$ be given, where $\sigma_i$ is $\rho_i$-Lipschitz and $\sigma_i(0) = 0$. Let spectral norm bounds $(s_1, \ldots, s_L)$, and matrix $(2,1)$ norm bounds $(b_1, \ldots, b_L)$ be given. Let data matrix $X \in \mathbb{R}^{n \times d}$ be given, where the $n$ rows correspond to data points. Let $\mathcal{H}_X$ denote*

the family of matrices obtained by evaluating $X$ with all choices of network $F_\mathcal{A} : \mathcal{H}_X :=$ $\left\{ F_\mathcal{A} \left( X^\top \right) : \mathcal{A} = (A_1, \ldots, A_L), \|A_i\|_\sigma \leq s_i, \left\| A_i^\top - M_i^\top \right\|_{2,1} \leq b_i \right\}$, where each matrix has dimension at most $W$ along each axis. Then for any $\epsilon > 0$,

$$\ln \mathcal{N} \left( \mathcal{H}_X, \epsilon, \| \cdot \|_F \right) \leq \frac{\|X\|_F^2 \ln \left( 2W^2 \right)}{\epsilon^2} \left( \prod_{j=1}^L s_j^2 \rho_j^2 \right) \left( \sum_{i=1}^L \left( \frac{b_i}{s_i} \right)^{2/3} \right)^3.$$

Then using Lemma E.1 and Assumption 3.1, we can obtain

$$\ln \mathcal{N} \left( \epsilon, \mathcal{F}_{\breve{\mathbf{X}}}, \| \cdot \|_F \right) \leq \frac{\|\breve{\mathbf{X}}\|_F^2 \ln(2\bar{d}^2) \rho^{4L-2} (2\vartheta)^2 d\kappa\zeta}{\epsilon^2}$$

where $\kappa = \alpha_f^{2L} \alpha_h^{2L}$ and $\zeta = \left( 1 + L \left( \frac{b_f}{\alpha_f} \right)^{2/3} + L \left( \frac{b_h}{\alpha_h} \right)^{2/3} \right)^3$. This finished the proof. $\qquad\square$

# F   Time Complexity Analysis

The notations used in the complexity analysis are explained as follows:

- $n, m$ to denote the number of samples of the training phase and inference phase, respectively.
- Missforest is a well-known data imputation algorithm based on random forest ($\mathcal{O}(t_1 \cdot v \cdot n \log n)$) where $t_1$ denotes the number of trees, $v$ denotes the number of attributes.
- $T, T_g, T_d, T_{ae}, T_{oc}$ denote the iterations of corresponding methods.
- $\bar{L}$ and $\bar{d}$ denote the number of layers of the neural network and the maximum width of the layers of the corresponding models, respectively.
- $t_2$ denotes the number of trees of I-Forest and $t$ is the maximum iterations of the Sinkhorn algorithm.
- $p, \psi, K$ denote the key parameters of the corresponding methods.

Table 5: The time complexity of training and inference.

| DI Methods | AD Methods | Time Complexity (Training) | Time Complexity (Inference) |
|---|---|---|---|
| MissForest $\mathcal{O}(T \cdot p(t_1 \cdot v \cdot n \log n))$ | I-Forest | $\mathcal{O}(T \cdot p(t_1 \cdot v \cdot n \log n) + t_2 \cdot \psi \log \psi)$ | $\mathcal{O}(p(t_1 \cdot v \cdot m \log n) + t_2 \cdot m \log \psi)$ |
| | Deep SVDD | $\mathcal{O}(T \cdot p(t_1 \cdot v \cdot n \log n) + (T_{ae} + T_{oc})(n\bar{d}^2\bar{L} + n))$ | $\mathcal{O}(p(t_1 \cdot v \cdot m \log n) + (m\bar{d}^2\bar{L} + m))$ |
| | NeutraL AD | $\mathcal{O}(T \cdot p(t_1 \cdot v \cdot n \log n) + T(n\bar{d}^2\bar{L} + n \cdot K))$ | $\mathcal{O}(p(t_1 \cdot v \cdot m \log n) + (m\bar{d}^2\bar{L} + m \cdot K))$ |
| | DPAD | $\mathcal{O}(T \cdot p(t_1 \cdot v \cdot n \log n) + T(n\bar{d}^2\bar{L} + n^2))$ | $\mathcal{O}(p(t_1 \cdot v \cdot m \log n) + (m\bar{d}^2\bar{L} + mn))$ |
| GAIN $\mathcal{O}((T_g + T_d)n\bar{d}^2\bar{L})$ | I-Forest | $\mathcal{O}((T_g + T_d)n\bar{d}^2\bar{L} + t_2 \cdot \psi \log \psi)$ | $\mathcal{O}(m\bar{d}^2\bar{L} + t_2 \cdot m \log \psi)$ |
| | Deep SVDD | $\mathcal{O}((T_g + T_d)n\bar{d}^2\bar{L} + (T_{ae} + T_{oc})(n\bar{d}^2\bar{L} + n))$ | $\mathcal{O}(m\bar{d}^2\bar{L} + (m\bar{d}^2\bar{L} + m))$ |
| | NeutraL AD | $\mathcal{O}((T_g + T_d)n\bar{d}^2\bar{L} + T(n\bar{d}^2\bar{L} + n \cdot K))$ | $\mathcal{O}(m\bar{d}^2\bar{L} + (m\bar{d}^2\bar{L} + m \cdot K))$ |
| | DPAD | $\mathcal{O}((T_g + T_d)n\bar{d}^2\bar{L} + T(n\bar{d}^2\bar{L} + n^2))$ | $\mathcal{O}(m\bar{d}^2\bar{L} + (m\bar{d}^2\bar{L} + mn))$ |
| ImAD (Ours) | | $\mathcal{O}(T(n\bar{d}^2\bar{L} + t \cdot n^2))$ | $\mathcal{O}(m\bar{d}^2\bar{L} + m)$ |

# G   Gain of Detection Performance from Pseudo-Abnormal Samples

## G.1   ImAD Benefits from Learning Pseudo-abnormal Samples

In this section, we explore the influences of introduced pseudo-abnormal samples for detection performance. On all datasets used in our experiments, we remove the pseudo-abnormal samples in the training process and only use incomplete normal data to train ImAD. The experimental results are shown in Table 6 and Table 7, where the detection performance of ImAD is improved in all the cases when introducing pseudo-abnormal samples into the training process. This indicates that the generated pseudo-abnormal samples are practical and effective for anomaly detection on incomplete data.

## G.2   ImAD's Pseudo-abnormal Samples Can Improve "impute-then-detect" Methods

Furthermore, we save the generated pseudo-abnormal data from the training process of ImAD on Titanic and Bladder, and then we add them into the training set for data imputation of "impute-then-detect" methods. The related results are provided in Table 8. We see that the pseudo-abnormal

samples learned by our ImAD can improve the performance of "impute-then-detect" methods. The reason is that with the pseudo-abnormal data, the imputation algorithms, i.e., MissForest and GAIN, generalize better on the test data. These results further confirm the effectiveness of the generative module of our ImAD.

Table 6: Gain of detection performance of ImAD from pseudo-abnormal samples on datasets with manually constructed missing values.

| Datasets | Settings | AUROC(%) | | AUPRC(%) | |
|---|---|---|---|---|---|
| | | mr=0.2 | mr=0.5 | mr=0.2 | mr=0.5 |
| Adult | ImAD w/o pseudo-abnormal samples | 65.30 (4.36) | 68.15 (5.66) | 67.49 (3.98) | 69.50 (4.35) |
| | ImAD | 76.51 (2.12) | 71.19 (1.63) | 73.42 (2.08) | 71.50 (2.02) |
| KDD | ImAD w/o pseudo-abnormal samples | 96.00 (2.03) | 91.50 (1.59) | 94.70 (1.03) | 92.09 (1.14) |
| | ImAD | 97.01 (0.33) | 90.78 (1.35) | 95.96 (0.18) | 91.58 (0.32) |
| Botnet | ImAD w/o pseudo-abnormal samples | 99.78 (0.05) | 99.38 (0.16) | 99.78 (0.05) | 99.40 (0.15) |
| | ImAD | 99.71 (0.22) | 99.53 (0.25) | 99.68 (0.24) | 99.58 (0.20) |
| Arrhythmia | ImAD w/o pseudo-abnormal samples | 78.28 (4.03) | 79.04 (0.73) | 76.98 (3.55) | 78.27 (1.88) |
| | ImAD | 82.24 (1.76) | 81.76 (1.19) | 83.74 (1.85) | 83.37 (1.36) |
| Speech | ImAD w/o pseudo-abnormal samples | 53.22 (3.62) | 47.28 (4.27) | 53.47 (4.66) | 49.92(3.13) |
| | ImAD | 61.94 (2.77) | 58.66 (1.40) | 60.43 (3.33) | 58.13 (1.48) |
| Segerstolpe | ImAD w/o pseudo-abnormal samples | 97.29 (0.93) | 96.53 (1.15) | 96.62 (0.90) | 96.60 (1.16) |
| | ImAD | 99.14 (0.88) | 96.86 (0.67) | 98.98 (1.18) | 96.85 (0.54) |
| Usoskin | ImAD w/o pseudo-abnormal samples | 79.07 (2.13) | 76.67 (1.27) | 79.43 (0.94) | 78.57 (2.48) |
| | ImAD | 84.95 (1.29) | 79.23 (2.49) | 85.48 (2.34) | 80.06 (3.40) |

Table 7: Gain of detection performance of ImAD from pseudo-abnormal samples on datasets with inherent missing values.

| Datasets | Settings | AUROC(%) | AUPRC(%) |
|---|---|---|---|
| Titanic | ImAD w/o pseudo-abnormal samples | 76.14 (1.66) | 75.46 (1.32) |
| | ImAD | 82.09 (0.99) | 81.39 (0.84) |
| MovieLens1M | ImAD w/o pseudo-abnormal samples | 60.62(1.35) | 60.29(1.44) |
| | ImAD | 66.32 (1.36) | 65.34 (1.35) |
| Bladder | ImAD w/o pseudo-abnormal samples | 99.90 (0.21) | 99.87 (0.29) |
| | ImAD | 1.00 (0.00) | 1.00 (0.00) |
| Seq2-Heart | ImAD w/o pseudo-abnormal samples | 95.19 (0.71) | 94.27 (1.18) |
| | ImAD | 96.62 (0.11) | 96.40 (0.19) |

Table 8: Gain of detection performance provided by the pseudo-abnormal samples for "impute-then-detect" methods.

| Settings | | Titanic | | Bladder | |
|---|---|---|---|---|---|
| Imputation Method | AD method | AUROC(%) | AUPRC(%) | AUROC(%) | AUPRC(%) |
| MissForest | IForest | 79.72 | 78.50 | 44.53 | 46.84 |
| MissForest with pseudo-abnormal data | IForest | 79.24 | 78.72 | 50.33 | 51.67 |
| GAIN | IForest | 79.46 | 78.69 | 45.77 | 47.62 |
| GAIN with pseudo-abnormal data | IForest | 79.91 | 79.47 | 50.29 | 52.83 |

# H  Influence of Constrained Radii $r_1, r_2$ for Detection Performance

In this section, we explore the influences of constrained radii $r_1, r_2$ for detection performance. We change the latent dimension $d = \{4, 8, 16, 32, 64, 128, 256, 512\}$ and carry out related experiments. Detailed experimental settings and results are provided in Table 9 and Figure 6 and Figure 7, respectively.

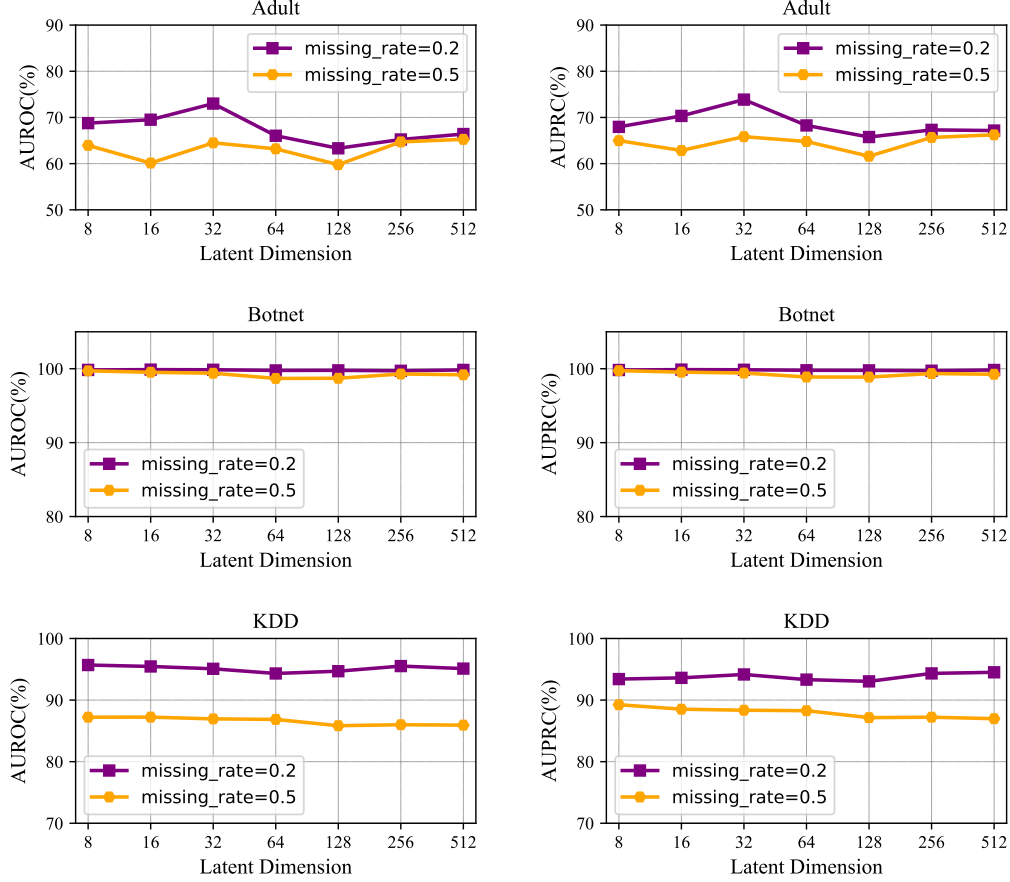

Figure 6: The detection performance on Adult, Botnet and KDD with different latent dimension.

As showed in Table 9, we change the dimension $d$ of latent space and then get $r = \sigma\sqrt{F_d^{-1}(p)}$ (See Proposition A.1) and set target distribution $\mathcal{D}_{\mathbf{z}} \sim \mathcal{N}(\mathbf{0}, 0.5^2 \cdot \mathbf{I}_d), \mathcal{D}_{\bar{\mathbf{z}}} \sim \mathcal{N}(\mathbf{0}, \mathbf{I}_d)$ and set $p = 0.9$.

Table 9: The constrained radii $r_1, r_2$ under with different latent dimensions.

| Radius | Latent Dimension ($d$) | | | | | | | |
|---|---|---|---|---|---|---|---|---|
| | 4 | 8 | 16 | 32 | 64 | 128 | 256 | 512 |
| $r_1 = 0.5\sqrt{F_d^{-1}(0.9)}$ | 1.39 | 1.82 | 2.42 | 3.26 | 4.44 | 6.10 | 8.45 | 11.76 |
| $r_2 = \sqrt{F_d^{-1}(0.9)}$ | 2.78 | 3.65 | 4.85 | 6.52 | 8.88 | 12.20 | 16.90 | 23.52 |

Figure 6 and Figure 7 shows the fluctuation of detection performance with different latent dimension $d$. It can be observed that our method is not very sensitive to changes in the radii $r_1$ and $r_2$, but its performance degrades with a reduction in the latent dimension. This is reasonable since the smaller latent dimension results in more information loss.

# I  Ablation Study and Sensitivity Analysis of Hyperparameters

For hyper-parameters $\alpha, \beta, \lambda$ used in our experiments, we vary them in a large range to analyze the sensitivity of ImAD under MCAR. For hyper-parameter $\beta$, it cannot be set to 0 because the imputation module is an indispensable part in the presence of missing values. The average results are shown in Figure 8.

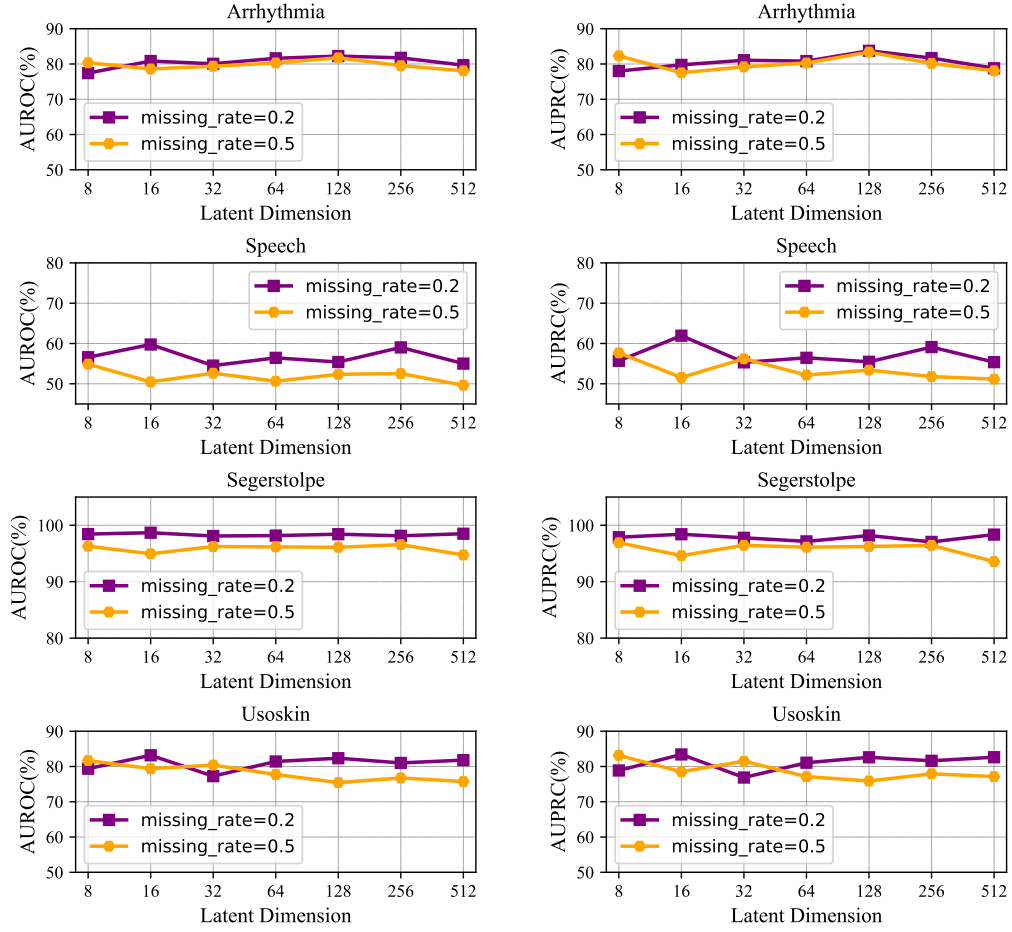

Figure 7: The detection performance on Arrhythmia, Speech, Segerstolpe and Usoskin with different latent dimension.

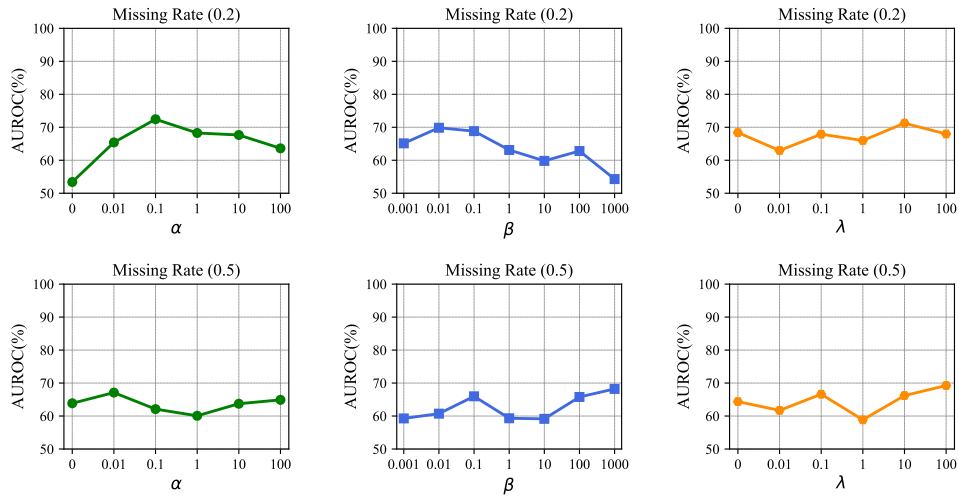

Figure 8: Sensitivity analysis of hyperparameters $\alpha, \beta, \lambda$ on Adult dataset.

# J Detailed Experimental Implementations

## J.1 Dataset Description

- **Adult**[2] [Becker and Kohavi, 1996] is from the 1994 Census Income database with 14 variables including both categorical and continuous variables. The samples of income $\leq$ 50K are regarded as normal data, and the samples of income > 50K are regarded as abnormal data. Data preparation follows the previous work [Han et al., 2023].
- **KDD**[3][Lichman, 2013] is the KDDCUP99 10 percent dataset from the UCI repository and contains 121 variables including both categorical and continuous variables. The attack samples are regarded as normal data, and the non-attack samples are regarded as abnormal data.
- **Arrhythmia**[4] [Rayana, 2016] is an ECG dataset. It was used to identify arrhythmic samples in five classes and contains 452 instances with 274 attributes.
- **Speech**[5] [Rayana, 2016] consists of 3686 segments of English speech spoken with different accents and is represented by 400-dimensional so-called i-vectors which are widely used state-of-the-art features for speaker and language recognition.
- **Segerstolpe**[6] [Segerstolpe et al., 2016] is a scRNA-seq dataset of human pancreas islets which includes six cell types: "alpha", "beta", "delta", "ductal", "endothelial" and "gamma". In our experiments, "alpha" is regarded as normal data and "beta" is regarded as abnormal data.
- **Usoskin**[7] [Usoskin et al., 2015] is a dataset employed for the analysis of sensory neuron cells, specifically originating from the mouse lumbar dorsal root ganglion. The dataset encompasses four distinct cell types: non-peptidergic nociceptor cells (NP), peptidergic nociceptor cells (PEP), neurofilament-containing cells (NF), and tyrosine hydroxylase-containing cells (TH). In our experiment, TH is regarded as normal data and PEP is abnormal data.
- **Botnet**[8] [Meidan et al., 2018] is a public botnet datasets for the IoT. it was gathered from 9 commercial IoT devices authentically infected by Mirai and BASHLITE. There are 7,062,606 instances in the original datasets. In our experiments, we use "Ecobee_Thermostat" subset of the original data, in which "benign_traffic" is regarded as normal data and "gafgyt_attacks" is regarded as abnormal data. The "gafgyt_attacks" has five attack types and we randomly select 1,000 samples from each type as abnormal data of the test set.
- **Titanic**[9] [Chen et al., 2023] is a classification dataset to detect the survival on the Titanic. In our experiments, we use nine features including "gender, ticket, cabin, age, sibsp, parch, fare, embarked and pclass". The instances that did not survive are considered normal samples and those that survived are considered abnormal(or unusual) samples.
- **MovieLens1M**[10] Han et al. [2021] contains 1,000,209 anonymous rating of approximately 3,900 movies made by 6,040 MovieLens users who joined MovieLens in 2000. Due to the missing rate of the original dataset is near 95%, we remove some columns with quite high missing rate and obtain a new dataset with 82% missing rate. Since the age of all samples is divided into seven groups, we chose the middle five groups (18 < age < 56) as normal samples and the remaining as abnormal samples.
- **Bladder**[11] is a cell transcriptome data from the model organism Mus musculus, in which contains 4 cell types (bladder cell, bladder urothelial cell, endothelial cell and leukocyte). We use the instance from "bladder cell" as normal samples and those from " leukocyte" as abnormal samples.
- **Seq2-Heart**[12] [Schaum et al., 2018] is a single cell transcriptome data from the model organism Mus musculus, containing nearly 100,000 cells from 20 organs and tissues. There are 8 cell types in this data. We use the instances with "fibroblast" type as normal samples and those with "myofibroblast" type as abnormal samples.

---

[2]https://archive.ics.uci.edu/dataset/2/adult

[3]https://kdd.ics.uci.edu/databases/kddcup99/

[4]http://odds.cs.stonybrook.edu/arrhythmia-dataset/

[5]https://odds.cs.stonybrook.edu/speech-dataset/

[6]https://cblast.gao-lab.org/download

[7]https://linnarssonlab.org/drg/

[8]https://archive.ics.uci.edu/dataset/442/detection+of+iot+botnet+attacks+n+baiot

[9]https://www.kaggle.com/c/titanic/data

[10]https://grouplens.org/datasets/movielens/1m/

[11]https://cblast.gao-lab.org/download

[12]https://cblast.gao-lab.org/download

## J.2 Missing Mechanisms

In this work, we evaluate the detection performance of all the baselines under the three distinct missing mechanisms and we follow the previous work [Muzellec et al., 2020] to set the missing value generation mechanism.

A detailed explanation of our implementation is provided as follows.

- **MCAR**: missing completely at random if the missingness is independent of the data. In our implementation, each entry is masked according to the realization of a Bernoulli random variable with parameter $p = \{0.2, 0.5\}$.
- **MAR**: missing at random if the missingness depends only on the observed values. In the MAR setting, for all experiments, a fixed subset of variables that cannot have missing values is sampled. Then, the entries from the remaining variables are masked according to a logistic model with random weights, which takes the non-missing variables as inputs. A bias term is fitted using line search to attain the desired proportion of missing values.
- **MNAR**: missing not at random if the missingness depends on both the observed values and the unobserved values. In the MNAR setting, first, we sample a subset of variables whose values in the lower and upper p-th percentiles are masked according to a Bernoulli random variable, and the values in-between are left not missing.

## J.3 Sampling in Target Distribution

In our experiments, we select two truncated Gaussian distribution $\mathcal{N}(\mathbf{0}, \sigma^2\mathbf{I}_d)$ with different $\sigma$ as target distribution $\mathcal{D}_{\mathbf{z}}, \mathcal{D}_{\bar{\mathbf{z}}}$ and set $\sigma = 0.5, \sigma = 1.0$ respectively. For target distribution $\mathcal{D}_{\mathbf{z}} \sim \mathcal{N}(\mathbf{0}, 0.5^2 \cdot \mathbf{I}_d)$, according to the Proposition A.1, we set constrained radius $r = 0.5\sqrt{F_d^{-1}(p)}$ where $d$ denotes the latent dimension and set $p = 0.9$. Similarity, for target distribution $\mathcal{D}_{\bar{\mathbf{z}}} \sim \mathcal{N}(\mathbf{0}, \mathbf{I}_d)$, we set $r_1 = r$ and $r_2 = \sqrt{F_d^{-1}(p)}$ and set $p = 0.9$.

## J.4 All Baselines

For the data imputation method used in our experiments, GAIN [13], MissOT [14], we use official code and the hyperparameters are fine-tuned as suggested in the original paper. For MissForest, we use *missingpy* [15] which is a library for missing data imputation in Python to implement the MissForest [Stekhoven and Bühlmann, 2012] algorithm. For anomaly detection method, Deep SVDD [16] [Rubin, 1976], NeutraL AD [17] [Qiu et al., 2021] and DPAD [Fu et al., 2024], we use official code and the hyperparameters are fine-tuned as suggested in the original paper. For Isolation Forest, we use *scikit-learn* [18] to implement the Isolation Forest [Liu et al., 2008] algorithm.

## J.5 Hyper-parameter Settings

The hyperparameters used in our experiments are provided in Table 10.

# K More Experimental Results

In this section, we conduct experiments on the Speech dataset with a missing rate mr $\in \{0.1, 0.2, 0.3, 0.4, 0.5, 0.6, 0.7, 0.8\}$. The related results are visualized in Figure 9, where the detection performance of "impute-then-detect" methods does not degrade and some of them even improve with the increasing of missing rate from 0.1 to 0.8. Moreover, our proposed method outperforms all baselines in almost all cases.

Table 10: Hyperparameters settings of the proposed method on all datasets.

| Datasets | Missing rate | Latent dimension | Learning rate | $\alpha$ | $\beta$ | $\lambda$ |
|---|---|---|---|---|---|---|
| Adult | mr=0.2 | 4 | 0.0002 | 5 | 20 | 1 |
| | mr=0.5 | 4 | 0.0002 | 1 | 10 | 2 |
| Botnet | mr=0.2 | 32 | 0.0001 | 1 | 1 | 1 |
| | mr=0.5 | 32 | 0.0001 | 1 | 1 | 1 |
| KDD | mr=0.2 | 32 | 0.0001 | 1 | 5 | 1 |
| | mr=0.5 | 32 | 0.0001 | 1 | 5 | 1 |
| Arrhythmia | mr=0.2 | 128 | 0.0001 | 1 | 1 | 1 |
| | mr=0.5 | 128 | 0.0001 | 1 | 1 | 1 |
| Speech | mr=0.2 | 128 | 0.0005 | 0.2 | 0.1 | 1 |
| | mr=0.5 | 128 | 0.0005 | 0.2 | 0.2 | 1 |
| Segerstolpe | mr=0.2 | 128 | 0.0001 | 1 | 1 | 1 |
| | mr=0.5 | 128 | 0.0001 | 1 | 1 | 1 |
| Usoskin | mr=0.2 | 128 | 0.0001 | 0.2 | 0.2 | 0.2 |
| | mr=0.5 | 128 | 0.0001 | 0.2 | 0.2 | 0.2 |
| Titanic | - | 4 | 0.0001 | 0.1 | 0.1 | 0.01 |
| MovieLens1M | - | 128 | 0.005 | 1 | 1 | 1 |
| Bladder | - | 128 | 0.0001 | 1 | 0.01 | 1 |
| Seq2-Heart | - | 128 | 0.0001 | 1 | 1 | 1 |

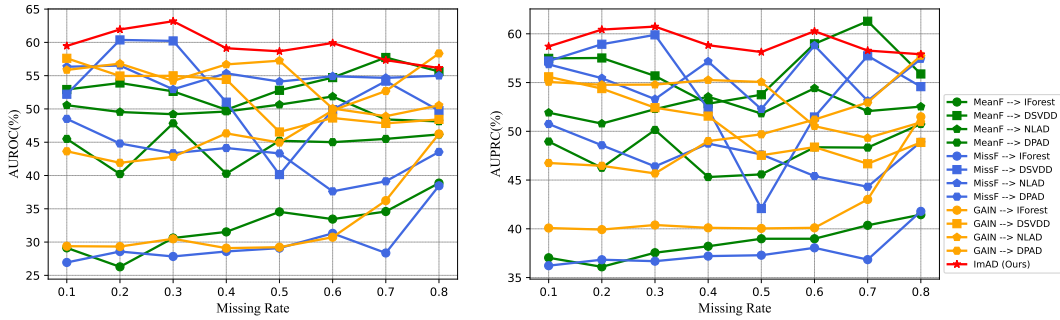

Figure 9: The performance fluctuation with the changes of missing rate from 0.1 to 0.8 on the Speech dataset. "MeanF" and "MissF" denotes Mean-Filling and MissForest, respectively.

In some real scenarios, it is possible that the missing rates of training and test sets are not equal. In this section, we conduct related experiments on the Speech dataset. In these experiments, we keep the missing rate mr = 0.5 on the training set and change the missing rate from 0.2 to 0.8 on the test set. We visualize the experimental results in Figure 10.

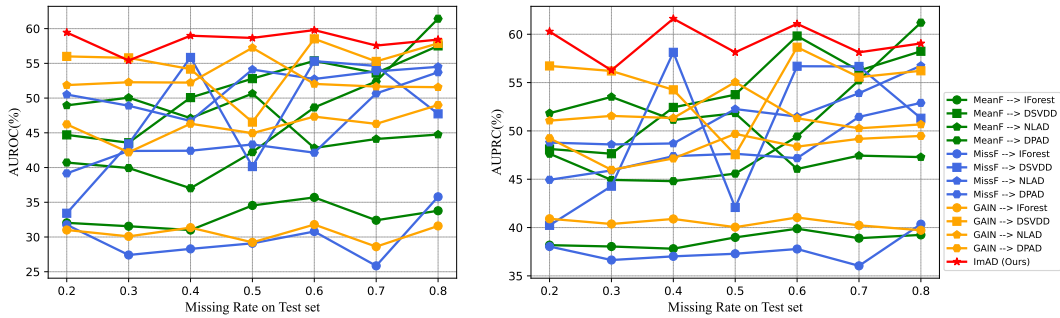

Figure 10: The performance fluctuation with the changes of missing rate from 0.2 to 0.8 on the test set of Speech. "MeanF" and "MissF" denotes Mean-Filling and MissForest, respectively.

With such an experimental setup, the performance of the methods based on Mean-Filling and MissForest fluctuates significantly. Our proposed method outperforms all baselines in almost all cases in spite of also showing some degree of performance fluctuation.

The experimental results on the Segerstolpe and Usoskin with MCAR are provided in Table 11. The experimental results on the Botnet dataset with MCAR are provided in Table 12. In addition to the missing mechanism MCAR, we also compare ImAD with "impute-then-detect" baselines under missing mechanism MAR and MNAR. Note that we did not employ GAIN [Yoon et al., 2018] under MAR and MNAR because GAIN was proposed under the assumption of MCAR. Instead, we utilize MissOT [Muzellec et al., 2020] as the imputation method under MAR and MNAR. For all baselines, the experimental results on the Adult dataset under MAR and MNAR are shown in Table 13 and Table 14, respectively, in which, our proposed method outperforms all two-stage baselines in all cases.

Table 11: Detection accuracy (AUROC and AUPRC (%, mean and std)) on Segerstolpe and Usoskin datasets with MCAR. The best result in each case is marked in **bold**.

| DI Methods | AD Methods | Segerstolpe | | | | Usoskin | | | |
| | | AUROC(%) | | AUPRC(%) | | AUROC(%) | | AUPRC(%) | |
| | | mr = 0.2 | mr = 0.5 | mr = 0.2 | mr = 0.5 | mr = 0.2 | mr = 0.5 | mr = 0.2 | mr = 0.5 |
|---|---|---|---|---|---|---|---|---|---|
| Mean-Filling | I-Forest | 94.48(1.93) | 91.23(3.22) | 95.09(2.18) | 91.77(3.49) | 44.49(5.41) | 38.53(2.93) | 46.57(5.26) | 41.61(2.29) |
| | Deep SVDD | 94.14(1.91) | 91.52(2.46) | 94.46(1.57) | 87.89(2.02) | 78.57(0.15) | 78.41(0.34) | 80.86(0.18) | 80.61(0.23) |
| | NeutraL AD | 98.66(0.74) | 94.93(1.80) | 98.15(1.34) | 94.02(2.98) | 69.95(2.74) | 66.73(4.59) | 75.07(2.34) | 69.63(4.60) |
| | DPAD | 98.78(0.66) | 96.02(0.83) | 98.36(1.34) | 94.77(1.93) | 81.95(1.57) | 80.23(1.14) | 83.27(0.89) | 82.38(0.84) |
| MissForest | I-Forest | 94.91 (1.35) | 96.68(0.79) | 95.94(1.23) | 97.56(0.59) | 45.19(4.56) | 49.64(7.43) | 46.97(3.04) | 49.74(5.64) |
| | Deep SVDD | 96.20(2.66) | 89.24(1.44) | 97.53(1.40) | 90.65(0.57) | 37.47(3.83) | 43.61(7.49) | 50.55(2.02) | 55.05(4.81) |
| | NeutraL AD | 97.89(1.45) | 89.38(2.80) | 97.71(1.76) | 84.61(3.78) | 57.43(4.59) | 53.74(2.27) | 63.65(2.40) | 61.05(4.16) |
| | DPAD | 97.09(0.74) | 95.31(1.36) | 94.93(1.05) | 93.75(1.79) | 81.25(1.93) | 80.52(1.80) | 83.58(1.32) | 82.69(1.02) |
| GAIN | I-Forest | 94.25(0.90) | 92.07(1.82) | 96.14(0.75) | 93.94(1.62) | 40.96(2.02) | 37.11(2.12) | 46.29(1.76) | 42.86(1.22) |
| | Deep SVDD | 92.46(4.25) | 94.32(1.93) | 92.25(2.40) | 92.88(1.26) | 49.99(5.69) | 65.48(2.94) | 54.85(1.61) | 64.54(0.74) |
| | NeutraL AD | 97.52(0.37) | 90.10(0.90) | 97.52(1.02) | 90.10(0.82) | 56.18(2.62) | 64.80(1.85) | 64.85(2.68) | 73.33(1.31) |
| | DPAD | 98.53(1.23) | **98.50**(1.25) | 97.42(2.52) | **97.73**(2.07) | 83.92(1.29) | 81.60(0.95) | 84.30(0.91) | 82.98(0.56) |
| ImAD (Ours) | | **99.14**(0.88) | 96.86(0.67) | **98.98**(1.18) | 96.85(0.54) | **84.95**(1.29) | **82.94**(1.79) | **85.48**(2.34) | **83.61**(1.39) |

Table 12: Detection performance in terms of AUROC and AUPRC (%, mean and std) on Botnet with MCAR. mr denotes the missing rate. The best result in each case is marked in **bold**. The results that exhibit an increase with the rising missing rate mr from 0.2 to 0.5 are emphasized by underlining.

| DI Methods | AD Methods | AUROC | | AUPRC | |
| | | mr = 0.2 | mr = 0.5 | mr = 0.2 | mr = 0.5 |
|---|---|---|---|---|---|
| Mean-Filling | I-Forest | 91.71(1.63) | 72.08(3.49) | 93.72(1.53) | 73.72(3.20) |
| | Deep SVDD | 50.07(0.55) | 57.87(0.95) | 61.19(0.43) | 63.60(0.68) |
| | NeutraL AD | 72.60(3.86) | 48.95(3.52) | 63.13(4.28) | 49.45(3.04) |
| | DPAD | 66.28(0.14) | 57.92(0.71) | 68.25(0.24) | 63.97(0.35) |
| MissForest | I-Forest | 95.72(0.96) | 93.86(0.70) | 97.25(0.69) | 95.68(0.52) |
| | Deep SVDD | 96.72(0.87) | 97.51(0.94) | 96.60(0.80) | 97.62(0.89) |
| | NeutraL AD | 99.04(0.26) | 97.27(0.59) | 98.92(0.24) | 97.68(0.53) |
| | DPAD | 81.98(0.60) | 76.74(0.60) | 85.03(0.26) | 78.25(0.30) |
| GAIN | I-Forest | 96.16(0.24) | 94.01(0.73) | 97.61(0.21) | 96.18(0.44) |
| | Deep SVDD | 98.68(0.11) | 98.02(0.41) | 98.35(0.14) | 97.59(0.46) |
| | NeutraL AD | 97.42(0.33) | **99.56**(0.27) | 96.89(0.36) | 99.41(0.35) |
| | DPAD | 99.55(0.71) | 97.63(0.01) | 99.33(0.85) | 96.92(0.04) |
| ImAD (Ours) | | **99.71**(0.22) | 99.53(0.25) | **99.68**(0.24) | **99.58**(0.20) |

# L   Limitations and Future Work

Anomaly detection and data imputation are ubiquitous tasks across various data types. In this work, we primarily focus on incomplete tabular data. However, other data types, such as image and time-series data, also need to be studied in similar scenarios. Therefore, we will conduct further studies on more data types in future work based on ImAD.

Table 13: Detection performance in terms of AUROC and AUPRC (% mean and std) on Adult with MAR. The best result in each case is marked in **bold**.

| DI Methods | AD Methods | AUROC | | AUPRC | |
|---|---|---|---|---|---|
| | | mr = 0.2 | mr = 0.5 | mr = 0.2 | mr = 0.5 |
| MissForest | I-Forest | 60.54(0.92) | 61.94(1.07) | 56.98(1.19) | 58.14(1.30) |
| | Deep SVDD | 61.53(6.24) | 56.22(7.75) | 65.49(3.30) | 61.76(3.16) |
| | NeutraL AD | 52.29(1.51) | 51.96(1.01) | 55.70(0.88) | 54.47(0.56) |
| | DPAD | 63.54(0.27) | 65.40(0.39) | 68.43(0.39) | 67.14(0.26) |
| MissOT$_{(MLP)}$ | I-Forest | 45.63(2.93) | 41.94(2.27) | 45.37(0.73) | 44.97(0.76) |
| | Deep SVDD | 51.68(4.17) | 39.59(6.95) | 53.71(3.95) | 54.13(3.36) |
| | NeutraL AD | 52.54(0.78) | 47.24(1.96) | 50.79(0.76) | 47.29(1.59) |
| | DPAD | 55.83(1.04) | 50.93(2.62) | 53.94(1.22) | 50.56(1.39) |
| ImAD (Ours) | | **77.43**(3.42) | **74.61**(2.18) | **75.07**(1.75) | **70.74**(1.21) |

Table 14: Detection performance in terms of AUROC and AUPRC (% mean and std) on Adult with MNAR. The best result in each case is marked in **bold**.

| DI Methods | AD Methods | AUROC | | AUPRC | |
|---|---|---|---|---|---|
| | | mr = 0.2 | mr = 0.5 | mr = 0.2 | mr = 0.5 |
| MissForest | I-Forest | 60.53(1.40) | 60.24(1.05) | 56.84(0.62) | 57.54(1.07) |
| | Deep SVDD | 54.90(7.71) | 57.54(4.33) | 63.74(2.27) | 62.48(2.19) |
| | NeutraL AD | 53.07(1.26) | 50.82(2.35) | 55.41(1.01) | 52.52(1.01) |
| | DPAD | 65.36(0.44) | 63.23(0.22) | 69.19(0.30) | 64.50(0.25) |
| MissOT$_{(MLP)}$ | I-Forest | 44.78(2.68) | 38.62(1.52) | 45.72(0.95) | 43.23(1.01) |
| | Deep SVDD | 45.77(8.47) | 50.29(6.28) | 54.04(2.57) | 50.90(2.43) |
| | NeutraL AD | 49.87(1.07) | 49.38(1.07) | 50.20(0.42) | 48.37(0.81) |
| | DPAD | 54.96(3.19) | 54.10(2.50) | 53.49(1.85) | 51.98(1.80) |
| ImAD (Ours) | | **73.73**(3.57) | **72.35**(1.53) | **71.60**(0.74) | **68.97**(0.31) |

